# CosAE: Learnable Fourier Series for Image Restoration

**Sifei Liu,    Shalini De Mello,    Jan Kautz**
NVIDIA
{sifeil, shalinig, jkautz}@nvidia.com

## Abstract

In this paper, we introduce Cosine Autoencoder (CosAE), a novel, generic Autoencoder that seamlessly leverages the classic Fourier series with a feed-forward neural network. CosAE represents an input image as a series of 2D Cosine time series, each defined by a tuple of learnable frequency and Fourier coefficients. This method stands in contrast to a conventional Autoencoder that often sacrifices detail in their reduced-resolution bottleneck latent spaces. CosAE, however, encodes frequency coefficients, i.e., the amplitudes and phases, in its bottleneck. This encoding enables extreme spatial compression, e.g., $64\times$ downsampled feature maps in the bottleneck, without losing detail upon decoding. We showcase the advantage of CosAE via extensive experiments on flexible-resolution super-resolution and blind image restoration, two highly challenging tasks that demand the restoration network to effectively generalize to complex and even unknown image degradations. Our method surpasses state-of-the-art approaches, highlighting its capability to learn a generalizable representation for image restoration. The project page is maintained at https://sifeiliu.net/CosAE-page/.

## 1 Introduction

Training an image Autoencoder by reconstruction is one of the most commonly adopted approaches for image representation learning. At the heart of such a learning process is creating an information bottleneck: the network first downsamples the input image to lower spatial dimensional features, then upsamples them back to reconstruct the input image. The key motivation of an information bottleneck is to enhance a network's ability to capture and preserve key data intrinsic patterns and structures, thereby representation and generalization, which is particularly vital in vision tasks where data quality is critical to performance. Introducing a narrow bottleneck in Autoencoder provides an effective way not only to learn general-purpose representations for recognition tasks [1, 2, 3, 4], but also to learn disentangled representations for image synthesis and manipulation [5, 6].

To establish an information bottleneck, most existing Autoencoder networks compress input images into a spatially compact latent space [1, 2, 3, 5, 6]. They learn mid- to high-level representation for classification and attributes disentanglement, by extracting the main shared structures while sifting out any noisy components. *However, a narrow bottleneck Autoencoder are rarely directly designed for image restoration task*, mainly because the use of a downsampled bottleneck in conventional Autoencoders often results in the loss of spatial details. Thus, these Autoencoders struggle to effectively represent high-frequency details, such as textures, compared to more coarse-grained structures. In other applications where pixel regression quality make a key role, e.g., VQVAE [6] or latent diffusion models (LDM) [7], researchers address this challenge by striking a balance between the spatial compactness of the bottleneck and its capability to preserve details. For instance, in training the Autoencoder [6] used for [6, 7], the downsampling stride is typically capped at $16\times$. Strides higher than this often result in considerable degradation of image quality [7].

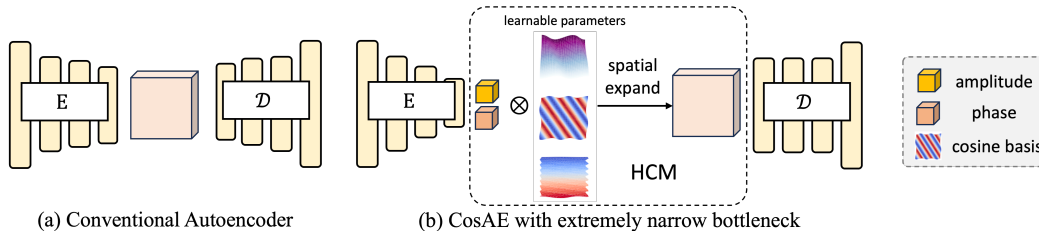

| | |
|---|---|
| (a) Conventional Autoencoder | (b) CosAE with extremely narrow bottleneck |

*Figure 1.* Unlike (a) a conventional Autoencoder, (b) CosAE encodes frequency domain coefficients as extremely narrow bottleneck feature maps. Via a basis construction module, it faithfully represents both the global structure and fine-grained details of an input image.

In this paper, we propose a novel Autoencoder directly applicable for image restoration – not only does it possess an extremely narrow bottleneck to boost representation capability, but it also faithfully preserves high-fidelity details. Our solution, the Cosine Autoencoder (CosAE), draws inspiration from Fourier transform principles, which demonstrates that finite signals can be depicted using a set of harmonic basis functions. We follow this principle to encode frequency domain coefficients, including amplitudes and phases, and integrate them with a set of Cosine basis functions before being decoded back to the original image (Figure. 1 (b)). This is achieved through a Harmonic Construction Module (HCM) that spatially expands the compact feature into a series of harmonic functions, following the formulation of the classical Fourier series. Unlike downsampled feature maps from the conventional Autoencoders, we found in CosAE, the representation with frequency coefficients can be extremely compressed spatially. Yet, these representations can effectively reconstruct fine-grained details when combined with high-frequency basis functions, e.g., the cosine basis in the 2nd row in Figure. 1.

As a generic Autoencoder, CosAE can be potentially utilized across various applications where traditional Autoencoders are applicable. In this paper, we demonstrate its capability for two challenging pixel reconstruction tasks: (a) flexible-resolution super-resolution, and (b) blind image restoration. We note that for tasks such as super-resolution, image denoising, and image enhancement, it is commonly believed that maintaining as many details as possible in the input without using a bottleneck will lead to better restoration [8, 9, 10]. However, the absence of an information bottleneck can potentially limit the capacity for representation. In contrast, the narrow bottleneck in CosAE encourages the alignment of the distributions of image representations under different types and levels of distortions. Unlike most previous work [11, 12] where distinct networks are needed for restoring images of different types of degradation, a single CosAE network demonstrates strong generalization capabilities across a wide range of degradation. We summarize our contributions as: (i) We propose CosAE, a novel generic Autoencoder featuring an extremely narrow bottleneck while preserving high-fidelity image details. (ii) We introduce a basis construction module, a key component of CosAE, that decomposes and represents not only the coarse structure but also the fine details of an image. (iii) CosAE can effectively restore images without the need for degradation types or upsampling ratios during inference. This capability makes it highly scalable in restoring images with unknown or complex types of degradation.

## 2 Related Work

**Autoencoder Networks.** Training Autoencoders [2] is a common approach for representation learning. By creating a bottleneck representation, Autoencoders abstract input data into generic visual representations. This method is used in Denoising Autoencoders (DAE)[1] for image restoration and Variational Autoencoders (VAE)[3] for image synthesis. Recent works have shown that Autoencoders can learn general representations for downstream tasks [13, 14, 15, 16, 4] and disentangled representations for image synthesis [17, 18, 5, 19]. While bottleneck representations have proven effective, recent works on RGB pixel reconstruction tasks [8, 9, 10] typically aim to maintain high spatial resolution during auto-encoding to preserve input details for better image reconstruction. In contrast, our work revisits the bottleneck concept, making it extremely narrow to achieve both generic representation and high-fidelity reconstruction.

**Image Restoration.** General image restoration aims to recover and enhance damaged or degraded images to improve their visual quality, clarity, and perceptual information. Within it, super-resolution [20, 21, 22, 23, 24, 8] and image denoising [25, 26, 27, 28, 29, 30] are the two major

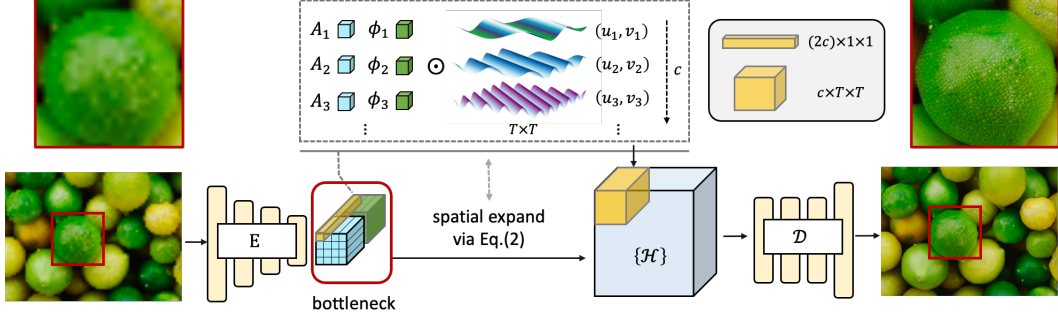

*Figure 2.* **Overview of CosAE.** CosAE contains an encoder that compresses an input image into a narrow bottleneck. Each bottleneck vector $[A_n, \phi_n]$ is translated into a group of learnable harmonic functions $\mathcal{H}$ of size $T \times T$, according to Eq. (2) (see the box). A decoder then reconstructs the input from the learned harmonic functions. Note $A_n$ equals to $A_{(u_n, v_n)}$ in Eq. (2) for simplicity.

sub-problems. Unlike conventional approaches that rely on specific corruption operators or separate networks for different levels of degradations, recent advancements have focused on learning more generalizable representations to handle complex and unknown degradations. Among them, flexible-resolution super-resolution [31, 32, 33] and blind image denoising [31, 34, 35] are widely studied. Our work focuses on these two applications. While most prior methods use encoders with minimal down-sampling, we show that a narrow bottleneck combined with continuous representation modules like LIIF [31] achieves consistent reconstruction across various corruption levels. In blind image restoration, recent networks rely on priors from pretrained models [36] or learnable dictionaries [37, 38], which complicate the training process and require additional measures to balance realism and fidelity. In contrast, CosAE maintains the simplicity of conventional Autoencoder training without such dependencies.

**Fourier Space.** Recently, methods like SIREN [39] and Fourier feature mapping [39, 40, 41, 42] have been used to preserve high-frequency details in networks. SIREN [39] utilizes the sinusoidal function as an activation layer to reconstruct high-frequency components, while other approaches [40, 41, 42] transform inputs into Fourier coefficients or feature maps to enhance detail retention. Frequency representations are proposed for texture synthesis [43, 19] and image super-resolution [32, 44]. Among image super-resolution works, the Fourier module of LTE [32] focuses on high-frequency residuals via a skip-link, limiting its capability to learn compressed frequency domain representations, while OPE-SR [44] employs a parameter-free decoder for pixel space decomposition using 1D Fourier basis, unlike CosAE, which uses a 2D Fourier basis to enable more compact information bottleneck representation.

## 3 Approach

Similar to conventional Autoencoders, CosAE has an encoder to produce a bottleneck latent representation (Sec.3.2) and a decoder to reconstruct the input image (Sec.3.5). Within the bottleneck, we introduce a harmonic construction module (HCM) (Sec.3.3) to spatially expand the latent representation into Fourier series (Figure.1 (b)). We start by revisiting the classic 2D Fourier transform theory [45] and its harmonic functions (Sec. 3.1). Furthermore, we delve into the details of the proposed HCM, showing how to integrate the classic Fourier transform into a latent space in neural networks.

### 3.1 Preliminary of Fourier Series

Fourier series is an effective tool to represent periodic or finite signals with a group of sinusoidal functions. Formally, the periodic extension of a finite 1D signal $\{x(t), t < T\}$ can be represented with Fourier series in the amplitude-phase form:

$$x(t) \approx \frac{A_0}{2} + \sum_{k=1}^{\infty} \mathcal{H}_k \tag{1}$$
$$\mathcal{H}(k, t) = A_k \cdot \cos(\frac{2\pi}{T} kt - \phi_k)$$

where $k$ denotes the discrete frequencies, $\mathcal{H}(k, t)$ denotes the $k^{th}$ harmonic component of the signal. Within $\mathcal{H}(k, t)$, we have the scalars $A_k$ and $\phi_k$ as the amplitude and the phase shift. Specifically, $A_0$

denotes the amplitude of the DC component. The Fourier series in Eq. (1) can be easily generalized to n-dimensional cases. In the context of a finite 2D signal, e.g., an image, the harmonic function with frequencies $(u, v)$ corresponding to the $x$ and $y$ dimensions is denoted as:

$$\mathcal{H}(u, v, x, y) = A_{(u,v)} \cdot \cos\left[\frac{2\pi}{T}(ux + vy) - \phi_{(u,v)}\right] \tag{2}$$

where $x < T$ and $y < T$. We denote $\cos[\frac{2\pi}{T}(ux + vy)]$ as a Cosine basis function, and the full term $\mathcal{H}(u, v, x, y)$ as harmonic function.

In our 2D domain, the amplitude and phase of an input image can be derived using the Fourier transform (e.g., 2D FFT) [45]. However, Fourier coefficients are neither learnable nor directly compressible for forming an information bottleneck, and thus cannot directly facilitate image restoration or visual representation learning tasks. Instead of applying FFT, we learn the amplitude and phase, denoted as $A_{(u,v)}$ and $\phi_{(u,v)}$, as bottleneck feature maps via an encoder, allowing flexible dimension configuration. To reconstruct the original image, we use Harmonic Component Modeling (HCM) to generate a set of learned harmonic functions (Eq. (2)), and a decoder acts as a learnable summation operator, mimicking the reconstruction process in Eq. (1).

The key insight is the compactness of Fourier coefficients in representing images, even those with intricate details. For example, a complex texture image may have only a few significant frequency components. By making Fourier coefficients a learnable representation, we can design an Autoencoder with an extremely narrow bottleneck.

## 3.2 Encoding Fourier Coefficients

The encoder of CosAE is responsible for learning Fourier Coefficients of the bottleneck latent space. Given a 2D square $P \times P$ image patch $I_p$, the encoder compresses the input signal into a bottleneck feature vector with the dimension of $2 \times c$, representing $c$ pairs of corresponding amplitudes and phases,

$$[A \in \mathcal{R}^c, \phi \in \mathcal{R}^c] = \mathbf{E}(I_p). \tag{3}$$

To generalize to images of any resolutions, in practice, we design the encoder with the downsampling stride setting as $P$, so that $A$ and $\phi$ are both $c$ channel-dimensional feature maps with its spatial resolution $1/P$ as that of the input image. In CosAE, a relatively large stride $P$ will be adopted so that the bottleneck can be very small: for face images, we maintain a downsampling stride as $P = 64$ (resulting in a 1D bottleneck of $\mathcal{R}^{2 \times c}$ for a $64 \times 64$ patch, see Eq. (3)), while for natural images, we use a stride as $P = 32$. Remarkably, CosAE is still able to faithfully reconstruct high-frequency details through combining these coefficients with the Cosine basis functions as latent feature maps, as introduced in the following.

## 3.3 Constructing Harmonic via HCM

We introduce a harmonic construction module (Figure. 2, middle box) that operates on the bottleneck to transform the predicted coefficients coming from the encoder into a set of 2D harmonic functions, i.e., $\mathcal{H}$ in Eq. (2). Taking the 2D square patch $I_p$ as an example. The encoder processes this patch to yield $c$ pairs of amplitudes $A \in \mathcal{R}^c$ and phases $\phi \in \mathcal{R}^c$. These are then combined with $c$ corresponding Cosine basis functions. Each basis is a $T \times T$ 2D Cosine waveform, where $T$ can be flexibly adjusted to achieve the desired output resolution. Thus, each pair of Fourier coefficient $A_{(u,v)}$ and $\phi_{(u,v)}$ will be spatially expanded to a $T \times T$ harmonic function, according to Eq. (2) (the spatial expansion is visualized as transitions between the two red blocks, in Figure. 2). The $c$ harmonic functions, formulated as a $T \times T \times c$ feature map, are then decoded to reconstruct the image patch $I_p$. Specifically, $T$ – the size of the Cosine basis functions can be customized during training, e.g., for the task of flexible-resolution super-resolution. We introduce the details in Sec. 4.1.

Notably, $T$ shouldn't be too small to formulate a valid 2D Cosine waveform. To balance with model efficiency, we set $T = P/2$, i.e., $T = 32$ for face images and $T = 16$ for natural images, where the decoder has an upsampling stride setting as 2. The $c$ pair of frequencies ($u \in \mathcal{R}^c, v \in \mathcal{R}^c$), which determine the Cosine basis functions, are also learnable, as detailed in Sec. 3.4.

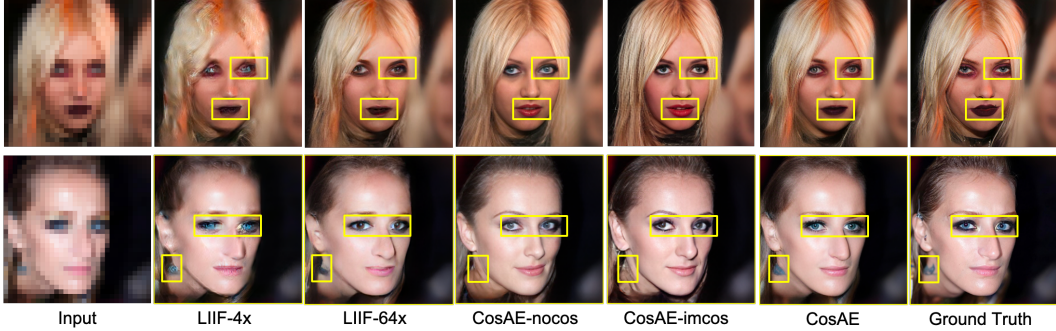

| Input | LIIF-4x | LIIF-64x | CosAE-nocos | CosAE-imcos | CosAE | Ground Truth |

*Figure 3.* **Qualitative evaluation for face FR-SR.** While models can super-resolve an input image with any ratio between 2 and 8, we show the **extreme case**: to upsample a $32 \times 32$ input to a $256 \times 256$ output, i.e., $\times 8$. We compared to both LIIF [10] and the CosAE variants stated at Sec. 4.2.

### 3.4 Learning Frequencies as Network Parameters

Many prior works learn the $u$ and $v$ also from an encoder [19, 32], similar to the Fourier coefficients, i.e., the frequencies are conditioned on the input signal. However, is it the optimal solution? To answer the question, we briefly revisit the properties of the harmonic function.

**Harmonics.** There are two key principles for setting frequencies in signal processing. First, frequencies $(u, v)$ in Eq. (5) are sampled in increments inversely proportional to the signal's length. Second, frequencies must satisfy the sampling theorem to avoid aliasing. To determine the range of the frequencies, we adopt the Fourier transform $F(\cdot)$ to the harmonic function (we use 1D case, and eliminate the amplitude and phase for simplicity),

$$
\begin{aligned}
F(\mathcal{H}_k(\omega)) &= \int_{-\infty}^{\infty} \cos(\tfrac{2\pi}{T} k t) e^{-i 2\pi \omega t} dt \\
&= \pi[\delta(2\pi(\omega - \tfrac{k}{T})) + \delta(2\pi(\omega + \tfrac{k}{T}))],
\end{aligned}
\tag{4}
$$

i.e., the bandwidth of the harmonic $B = k/T$. According to the Nyquist-Shannon theorem, for a sampling rate $f_s$, perfect reconstruction is guaranteed for a band-limit $B = \frac{k}{T} < \frac{f_s}{2}$. For discrete signals, such as images, where the sampling interval is $f_s = 1$ (pixel), $k$ can be restricted to $k < \frac{T}{2}$ to prevent aliasing and ensure an accurate harmonic representation.

**Our solution.** Based on the design of the classic Discrete Fourier Transform (DFT) described above, we initialize $c$ pairs of $(u, v)$ within the range $[0, T/2]$. However, for more flexible designs, the $(u, v)$ set should adapt to different needs, such as increasing frequency pairs for higher resolution or reducing them for efficiency. In such cases, uniform sampling can result in sparse coverage. For example, with $T = 64$ and the number of basis maps set to 64, both $u$ and $v$ are sparse, sampled as $[0, 4, 8, \ldots, 32]$. Since natural image frequencies are not uniformly distributed, we make $(u, v)$ learnable parameters shared across all input images, enabling effective modeling of diverse frequencies during training. Compared to prior approaches that adapt frequencies based on individual input images [32, 19], our design aligns more closely with classic principles by using globally shared frequencies that span a full range. Such design also addresses a common limitation in previous models emphasizing lower frequencies that dominate natural images, allowing for a more balanced and comprehensive frequency representation.

### 3.5 Decoding Harmonic for Reconstruction

Once the basis $\{\mathcal{H}\}_{T \times T \times c}$ are obtained, we introduce a decoder network $\mathcal{D}$ to map them to RGB pixels:

$$
X = \mathcal{D}(\{\mathcal{H}(u, v, x, y)\}); \ (u, v) < T/2, (x, y) < T
\tag{5}
$$

The architecture can be flexibly designed. E.g., our framework allows for either implicit representation networks [10, 39, 32, 33], sampling only a small portion of pixels to decode during training, or convolutional/transformer-based decoders that use the full latent space, accommodating additional modules like discriminators for full-sized outputs.

*Table 1.* **FR-SR evaluation on the val splits of FFHQ+CelebA faces (15k images)**. We upsample LR images of different input resolution (row-2), all to $256 \times 256$. The upper block displays methods using only L1 loss, while the lower part show those trained with full objectives, denoted with (G). The gray block in the bottom presents results for ablation studies (Sec. 4.2).

| | FID-15k↓ | | | | LPIPS↓ | | | | PSNR (db)↑ | | | | SSIM↑ | | | |
|---|---|---|---|---|---|---|---|---|---|---|---|---|---|---|---|---|
| LR-res | 32 | 48 | 64 | 128 | 32 | 48 | 64 | 128 | 32 | 48 | 64 | 128 | 32 | 48 | 64 | 128 |
| LIIF-64x | | N/A | | | 0.41 | 0.38 | 0.37 | 0.37 | 26.03 | 25.23 | 23.95 | 26.37 | 0.68 | 0.71 | 0.73 | 0.74 |
| LTE-64x | | N/A | | | 0.48 | 0.39 | 0.33 | 0.21 | 22.94 | 24.80 | 26.07 | 29.23 | 0.62 | 0.69 | 0.73 | 0.84 |
| ITNSR-64x | | N/A | | | 0.47 | 0.39 | 0.36 | 0.36 | 23.10 | 25.24 | 26.05 | 26.30 | 0.63 | 0.70 | 0.72 | 0.73 |
| **CosAE** | | N/A | | | **0.36** | **0.32** | **0.28** | 0.26 | **24.34** | **26.18** | **27.52** | 28.51 | **0.70** | **0.75** | **0.79** | 0.81 |
| LIIF-4x (G) | 34.93 | 18.80 | 11.89 | 6.23 | 0.35 | 0.27 | 0.22 | 0.14 | 22.94 | 24.68 | 26.19 | 27.94 | 0.63 | 0.70 | 0.75 | 0.80 |
| LIIF-64x (G) | 18.91 | 15.50 | 15.05 | 13.94 | 0.30 | 0.27 | 0.26 | 0.25 | 22.59 | 23.24 | 23.59 | 23.57 | 0.62 | 0.64 | 0.65 | 0.65 |
| **CosAE (G)** | 12.81 | **8.67** | **8.12** | 7.86 | **0.24** | **0.19** | **0.17** | **0.14** | 23.65 | 25.37 | 26.66 | 27.51 | **0.67** | **0.72** | **0.76** | 0.78 |
| CosAE-nocos (G) | 13.1 | 13.44 | 13.54 | 12.75 | 0.31 | 0.30 | 0.29 | 0.29 | 20.92 | 21.24 | 21.35 | 21.34 | 0.57 | 0.58 | 0.59 | 0.59 |
| CosAE-imcos (G) | 10.85 | 10.81 | 10.7 | 10.72 | 0.31 | 0.30 | 0.29 | 0.29 | 20.34 | 20.64 | 20.77 | 20.77 | 0.56 | 0.57 | 0.57 | 0.57 |
| CosAE-FT (G) | 16.55 | 12.68 | 11.78 | 10.70 | 0.26 | 0.22 | 0.19 | 0.16 | 23.55 | 25.00 | 25.90 | 26.30 | 0.66 | 0.70 | 0.73 | 0.74 |

## 3.6 Network Implementation

**FFT for Additional Input.** We found that incorporating the 2D FFT of the input image as extra channels enhances performance, as it intuitively aligns with the encoder's task of capturing Fourier coefficients in the latent space. Specifically, we apply the FFT to generate real and imaginary 2D maps for each RGB channel, which are then concatenated with the original RGB channels, yielding a 9-channel input map for the network.

**Network architectures.** We build our encoder and decoder based on [6], which has been widely utilized for auto-regression and diffusion models [6, 7, 46]. To enable a larger downsampling stride in the encoder, we add extra ResNet blocks and position the attention layers near the bottleneck. Correspondingly, we employ a relatively lightweight decoder with an upsampling factor of 2. We train CosAE on $256 \times 256$ patches for all the downstream tasks, while setting $c = 256$ for all the experiments. Please refer to Sec. 3.2 and 3.3 for other hyper-parameters.

Notably, we process each bottleneck feature vector independently, allowing inference on images of any size. This is achieved by replacing global attention with window attention [47] within each $T \times T$ feature cell, in the decoder. Likewise, during inference, for inputs with a different size than $256 \times 256$, we employ window attention as well on each non-overlapping $256 \times 256$ region.

**Objectives.** We follow [6] to train CosAE with L1 loss and a perceptual loss [48] as the reconstruction objectives, denoted as $\mathcal{L}_{rec} = \mathcal{L}_{l1} + \mathcal{L}_{lpips}$. In addition, we apply a patch-based discriminator [49], denoted as $\mathcal{L}_{GAN}$, to enhance the visual quality of the reconstructed image. The entire network, along with the $c$ pairs of learnable frequencies $\{u, v\} \in \mathcal{R}^{2 \times c}$, is trained with $\mathcal{L} = \mathcal{L}_{rec} + \lambda \mathcal{L}_{GAN}$. We keep the same settings for the adaptive weight $\lambda$ as [6]. More details are presented in the supplementary material.

# 4 Experimental Results

In this paper, we focus on the applications of flexible-resolution super-resolution and blind image denoising, leaving other potential auto-encoder-related applications for future research. We evaluate both tasks in the domains of face and natural images.

For both face and natural image domains, we train the CosAE with the upscaling factor from 1 to 8. During training, we sample a high-resolution (HR) image $I_{HR} \in \mathcal{R}^{H \times W \times 3}$ from the training set, and downsample it with a uniformly sampled ratio $r \in (1, 8]$ to obtain the low-resolution (LR) input image $I_{LR} \in \mathcal{R}^{\frac{H}{r} \times \frac{W}{r} \times 3}$. To train a super-resolution model with flexible output resolutions, we then sample an upscaling ratio as $u \in (1, r]$ to let CosAE upscale the input image to $I_{out} \in \mathcal{R}^{\frac{u}{r} H \times \frac{u}{r} W \times 3}$. This is conducted by setting $T = \frac{u}{r} T_{max}$, where $T_{max}$ refers to the $T$ in Sec. 3.3, which upscales the LR image to the original size of the HR image. Note that we always (bilinearly) resize the LR image to the original size of the HR image to fit the Encoder of CosAE. The size of the HR images is fixed to $256 \times 256$ during training, achieved either by resizing (e.g., for face images) or by random cropping (e.g., for natural images).

Our experiments also indicate that strictly limiting frequencies $(u, v)$ to prevent aliasing is NOT essential for effective performance. To validate this, we performed an ablation study (see Ap-

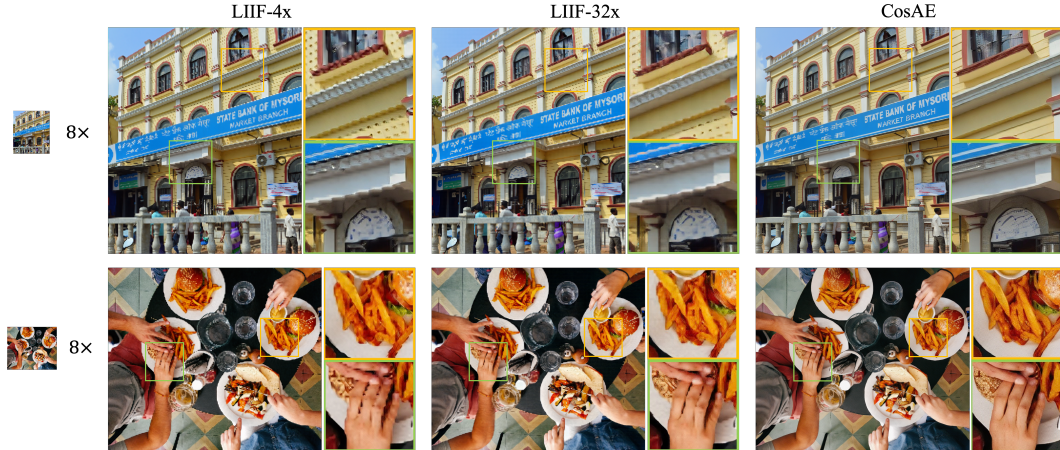

Figure 4. **FR-SR on DIV2K [24]**. We showcase $8\times$ results similar to Figure. 3. CosAE faithfully reconstructs fine details, for both regular (buildings) and irregular (skin and food) textures. Zoom in to see details.

pendix B.2) where we introduced a channel dropout variant during training, strictly constrained $\min(u, v) < T/2$, and observed the resulting outcomes. Further **frequency analysis** and discussions on **limitations**, including out-of-distribution generalization and common artifacts, are provided in the Appendix.

## 4.1 Flexible-resolution Super-resolution (FR-SR)

FR-SR has been intensively explored via implicit representation [10, 32, 33], in which the output resolution can be flexibly configured by setting the size of the coordinate map.

### 4.1.1 FR-SR for Face Images.

We train the face model using cropped faces from the training splits of FFHQ [50] and CelebA-HQ [51, 52], utilizing the official training and validation splits. To evaluate the output images, we use FID [53] and LPIPS [48] metrics, along with PSNR and SSIM to compare with methods that use pixel regression objectives [10, 32, 33].

**Baselines.** We compare CosAE with local implicit representation methods LIIF [10], LTE [32], and ITNSR [33], which generate arbitrary-resolution outputs by predicting RGB values at specific coordinates. For a fair comparison, we re-implemented these methods using the same Encoder as CosAE to ensure consistent representation capacity across models. Among those, LIIF is a direct counterpart to our approach, approximating the replacement of high-frequency components (Cosine basis functions) with plain coordinate maps. To ensure a fair comparison, we align LIIF by replacing its implicit decoder with ours (CNN with self-attention) and adopting the GAN loss.

To compare with LTE [32] and ITNSR [33], we replace their Encoders with ours but keep their original Decoders intact as they are key components. We do not add GAN loss as it does not enhance results. LTE does not have a narrow bottleneck due to a skipped link before the output. We align the input settings for all methods to match ours while keeping other components as per their original versions, denoting the modified networks as "-64x" in Table 1. Notably, during inference, CosAE performs blind super-resolution without needing the upsampling ratio, unlike other methods.

**Comparison with the State-of-the-Art Methods.** We demonstrate significantly improved performance with CosAE, compared with the baselines of LIIF-64x, LTE-64x and ITNSR-64x, both qualitatively (Figure. 3) and quantitatively (Table. 1). In Table. 1, networks trained only with L1 loss are placed in the upper block and those trained with full objectives (Sec. 3.6) are in the bottom rows. Our results are significantly better, especially in terms of large-ratio (e.g., $\times 8$ and $\times 6$) super-resolution, on all the matrices.

### 4.1.2 FR-SR for Natural Images.

To perform FR-SR training on natural images, we initially pretrain CosAE on the ImageNet [54] training split, using the same approach as the face model. However, we encountered distortions in the reconstructed images when applying the $4 \times 4$ bottleneck – same as the face model. To

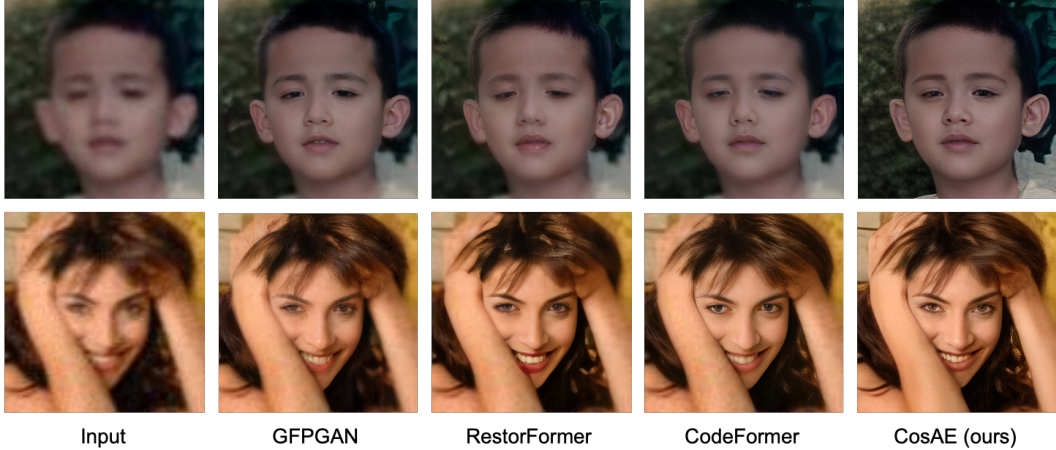

| Input | GFPGAN | RestorFormer | CodeFormer | CosAE (ours) |

*Figure 5.* **Comparisons with the STOA face restoration methods.** Results are shown for both **real-world** (WebPhoto-Test, 1st row) and synthetic degraded images (CelebA-Test, 2nd row). CosAE restores more high-frequency and realistic facial details compared to other methods.

address this, we adjusted the bottleneck to $8 \times 8$ with $T_{max} = 16$. Subsequently, we fine-tuned the model using the same training settings on a combination of the DIV-2K [24] training set and all the images from Flicker2K [55] (referred to as the DF2K dataset). We compared the CosAE with LIIF [10] aligned in the same way as training the face model, denoted as "LIIF-32x" as the Encoder for natural images has a $32\times$ downsampling stride. Both models were trained with the same datasets, input settings, and objectives.

To evaluate the performance, we used LPIPS on the DIV-2K validation set, considering the adoption of the discriminator. The FID score [53] is not applicable due to the testing set is small. We keep the same super-resolution up-scaling factor as that of the face (in Table. 2 we present the down-sampling factor instead of the resolution, given images of DIV2K have different sizes between individuals). Similarly, CosAE demonstrates significantly better performance than the others. Qualitative results are shown in Figure. 4, and Figure. 9, 10 in the Appendix.

### 4.2 Ablation Studies

To validate the design choices, we perform the Ablation Studies on the task of FR-SR for face images, which focus on the following aspects.

**W or w/o Cosine Basis Functions.** To evaluate if the Cosine basis functions play a key role in improving the performance, other than LIIF that replaces them with coordinate maps, we further replace them with plain upsampling layers, while preserving all the other components and training recipes. We denote it as *nocos*.

**Encoding Input with Cosine Basis.** To see if the Cosine basis functions can be placed other than the bottleneck, we encode the input image with Cosine basis functions, i.e., similar to Fourier transform to multiply each basis with the signal, but without the summation operator. We keep the bottleneck the same design as the *nocos* model. We denote it as *imcos*.

**Fourier Transform.** Instead of using the Decoder to mimic the summation operation in Eq. (1), we directly sum over the frequencies, i.e., the channel dimension of the bottleneck, before the Decoder. This is equivalent to performing the Fourier transform, instead of preserving the Fourier series, in the bottleneck latent space. We denote it as *FT*.

**Results.** We compare the ablated models with the others trained with full objectives, in Table. 1. Compared to CosAE, all the variants show degraded performance on all the matrices. We observe *nocos* and *imcos* exhibit visually pleasant faces with rich details, but keep less fidelity, e.g., the eyelid or lip color may completely changed, see Figure. 3. *FT* shows lower quality local region details given it does not fully leverage the capabilities of the decoder (see Figure. 12 in Supp., due to space limitation).

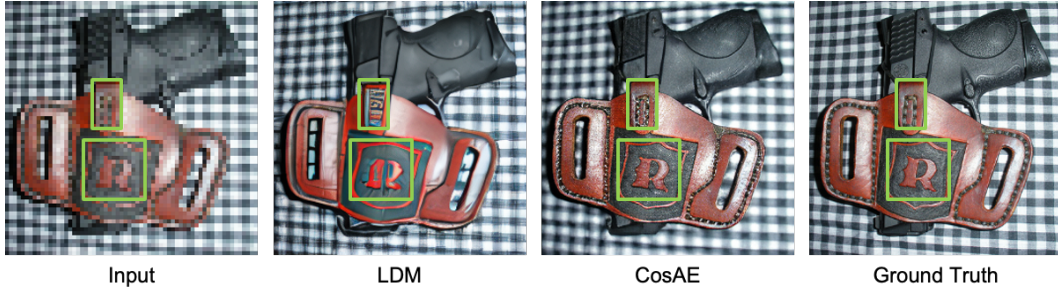

| Input | LDM | CosAE | Ground Truth |

*Figure 6.* Comparisons with LDM on $4\times$ **super-resolution on ImageNet**. In comparison, the CosAE produces more fine-grained details, in terms of both the structure and the texture. Zoom in to see details.

*Table 2.* **FR-SR evaluating of FR-SR on DIV2K**, e.g., $\times 2$ to $\times 8$, with LPIPS $\downarrow$ on the validation set (FID is N/A due to no reference set).

| up-ratio | $\times 8$ | $\times 5.33$ | $\times 4$ | $\times 2$ |
|---|---|---|---|---|
| LIIF-4x (G) | 0.41 | 0.33 | 0.30 | 0.26 |
| LIIF-32x (G) | 0.42 | 0.34 | 0.29 | 0.21 |
| CosAE (G) | **0.33** | **0.25** | **0.18** | **0.11** |

*Table 3.* **Quantitative comparisons on CelebA-Test.** Our CosAE shows better performance on almost all the metrics, except for the FID score (see the possible reasons in Sec. 4.3).

| Method | FID↓ | PSNR↑ | SSIM↑ | LPIPS↓ | IDD↓ |
|---|---|---|---|---|---|
| Input | 132.69 | 24.96 | 0.66 | 0.50 | 0.93 |
| DFDNet | 52.92 | 24.10 | 0.61 | 0.45 | 0.76 |
| PULSE | 67.75 | 21.61 | 0.63 | 0.46 | 1.20 |
| GFP-GAN | 42.39 | 24.46 | 0.67 | 0.36 | 0.60 |
| RestoreFormer | 41.45 | 24.42 | 0.64 | 0.36 | 0.57 |
| CodeFormer | 52.43 | 22.18 | 0.61 | 0.30 | 0.60 |
| CosAE | 52.03 | **25.65** | **0.70** | **0.27** | **0.53** |

**Wide or Narrow Bottleneck?** Is a narrow bottleneck better? Since CosAE does not support a wider bottleneck variant due to a certain size $T$ needs to be maintained for valid Cosine functions (Sec. 3.3). Thus, we modify a "wide" LIIF [10] by removing the downsampling layers, resulting in the *LIIF-4x* variant with a stride of 4. We evaluated *LIIF-4x* on face (Table 1, Figure.3) and natural images (Table2, Figure.4) using full objectives. Notably, the narrow bottleneck provided consistent performance across different ratios. Table1 shows minimal performance drop for super-resolving images by factors of 2 to 8 for *LIIF-64x* and other methods, compared to *LIIF-4x*. This consistent performance in FR-SR demonstrates the potential benefits of using a narrow bottleneck encoder.

## 4.3 Blind Face Image Restoration

Due to the space limitation, we present the results for blind face image restoration, while leaving details of blind natural image restoration in the supplementary material.

Blind image restoration restores high-quality images from complex and unknown degradation. For face images, most recent works [36, 37, 38] focus on two aspects: synthesis of degraded images that mimic the divergence and complex type of degradation via math operators [36, 56], and improve network architectures. We focus on the latter aspect.

For blind face image restoration, we adopt the degradation operators of [38] to synthesize degraded/-clean paired data during training. Instead of training the network from scratch, we found finetuning the CosAE network trained for the task of FR-SR (Sec. 4.1.1) yields faster convergence. Unlike training a FR-SR task, we instead fix $T = T_{max}$, and finetune the model with images of $512 \times 512$.

**Comparison to the State-of-the-Art Methods.** We compare the CosAE with the state-of-the-art methods, including DFDNet [34], PULSE [57], GFP-GAN [36], RestoreFormer [37] and Code-Former [38]. Following [36, 38], we evaluate CosAE in the CelebA-Test, LFW-Test, WebPhoto-Test, and WIDER-Test [38] datasets. We provide quantitative evaluations on both the synthetic dataset – the CelebA-Test dataset in Table. 3, and the real datasets including LFW-Test, WebPhoto-Test, and WIDER-Test in Table. 7 in Supp. In addition, we show qualitative results on both synthetic and real-world datasets in Figure. 5.

Both quantitative comparisons demonstrate the superior performance of our CosAE: it outperforms the other approaches on almost all the metrics, except for the FID score. However, it is important to note that the FID score may not provide an accurate assessment of realism when dealing with a relatively small test set, especially in the aforementioned real datasets. Qualitatively, compared to

*Table 4.* $\times 4$ **super-resolution on ImageNet-val.** CosAE achieves competitive realism scores compared with all the diffusion- and GAN-based methods [7, 58, 59, 60, 61, 62] with much larger capacities. 'N/A' indicates not reported.

| method | U-Net | SR3 | LDM-4 | emphLDM-4 | LDM-4-G | GigaGAN | $I^2$SB | DDRM | IRSDE | ResShift | CosAE |
|---|---|---|---|---|---|---|---|---|---|---|---|
| LPIPS↓ | N/A | N/A | N/A | N/A | N/A | N/A | 0.206 | 0.471 | 0.304 | 0.126 | 0.167 |
| FID-50k↓ | 15.2 | 5.2 | 2.8 | 2.4 | 4.4 | 1.2 | 2.8 | N/A | N/A | N/A | 2.1 |
| Params (M) | 625 | 625 | 169 | 552 | 183 | 359 | 552 | 552 | 137 | 121 | 67 |
| Inference (s) | N/A | N/A | N/A | N/A | N/A | 0.13 | 1.832 | 1.184 | 5.927 | 0.105 | 0.036 |

other approaches, CosAE synthesizes much more realistic facial details, e.g., in the hair and skin regions, as well as more wrinkles, freckles, beards, etc., than to generate a smoother one.

### 4.4 Fixed-ratio Super-resolution on ImageNet

In order to evaluate our approach against conventional methods for learning image representation, we conducted super-resolution experiments on the ImageNet dataset using a $4\times$ upsampling factor. To accomplish this, we finetuned the CosAE network trained with the FR-SR on natural images with a fixed $T = T_{max}$, i.e., no flexible output resolution being configured during training. This setting was applied to ImageNet images that were resized to $256 \times 256$ pixels. We compared CosAE with U-Net regression [7], LDM [7] series, GigaGAN [58], other diffusion-based models, including $I^2$SB [59], DDRM [60], IRSDE [61] and ResShift [62]. We use FID score with the reference of 50k validation images in ImageNet, and the LPIPS [48] as the evaluation matrices. In addition, model size as well as the inference speed (ours are tested on a single V100) are reported. As shown in table 4, CosAE outperforms most models in both efficiency (i.e., number of parameters and runtime) and performance. Qualitative visual comparison with LDM-4 [7] are shown in Figure. 14.

## 5 Conclusion

We introduce CosAE, a novel auto-encoder architecture for blind image restoration tasks. By incorporating learnable harmonic functions in the bottleneck, CosAE effectively captures underlying structures and patterns of images with extremely compressive representation, achieving accurate reconstruction with high-fidelity details. CosAE outperforms state-of-the-art methods in flexible-resolution super-resolution and blind image restoration. As a generic denoising Autoencoder, CosAE offers potential for various computer vision tasks.

## Footnotes

[1]Sourced from https://github.com/CompVis/latent-diffusion.

## References

[1] Pascal Vincent, Hugo Larochelle, Yoshua Bengio, and Pierre-Antoine Manzagol. Extracting and composing robust features with denoising autoencoders. In *Proceedings of the 25th international conference on Machine learning*, pages 1096–1103, 2008.

[2] Geoffrey E Hinton and Ruslan R Salakhutdinov. Reducing the dimensionality of data with neural networks. *science*, 313(5786):504–507, 2006.

[3] Diederik P Kingma and Max Welling. Auto-encoding variational bayes. *arXiv preprint arXiv:1312.6114*, 2013.

[4] Kaiming He, Xinlei Chen, Saining Xie, Yanghao Li, Piotr Dollár, and Ross Girshick. Masked autoencoders are scalable vision learners. In *Proceedings of the IEEE/CVF Conference on Computer Vision and Pattern Recognition*, pages 16000–16009, 2022.

[5] Taesung Park, Jun-Yan Zhu, Oliver Wang, Jingwan Lu, Eli Shechtman, Alexei A. Efros, and Richard Zhang. Swapping autoencoder for deep image manipulation. In *Advances in Neural Information Processing Systems*, 2020.

[6] Patrick Esser, Robin Rombach, and Björn Ommer. Taming transformers for high-resolution image synthesis, 2020.

[7] Robin Rombach, Andreas Blattmann, Dominik Lorenz, Patrick Esser, and Björn Ommer. High-resolution image synthesis with latent diffusion models, 2021.

[8] Bee Lim, Sanghyun Son, Heewon Kim, Seungjun Nah, and Kyoung Mu Lee. Enhanced deep residual networks for single image super-resolution. In *Proceedings of the IEEE conference on computer vision and pattern recognition workshops*, pages 136–144, 2017.

[9] Yulun Zhang, Yapeng Tian, Yu Kong, Bineng Zhong, and Yun Fu. Residual dense network for image super-resolution. In *Proceedings of the IEEE conference on computer vision and pattern recognition*, pages 2472–2481, 2018.

[10] Yinbo Chen, Sifei Liu, and Xiaolong Wang. Learning continuous image representation with local implicit image function. *arXiv preprint arXiv:2012.09161*, 2020.

[11] Kai Zhang, Wangmeng Zuo, Yunjin Chen, Deyu Meng, and Lei Zhang. Beyond a gaussian denoiser: Residual learning of deep cnn for image denoising. *IEEE transactions on image processing*, 26(7):3142–3155, 2017.

[12] Stamatios Lefkimmiatis. Non-local color image denoising with convolutional neural networks. In *Proceedings of the IEEE conference on computer vision and pattern recognition*, pages 3587–3596, 2017.

[13] Pascal Vincent, Hugo Larochelle, Isabelle Lajoie, Yoshua Bengio, Pierre-Antoine Manzagol, and Léon Bottou. Stacked denoising autoencoders: Learning useful representations in a deep network with a local denoising criterion. *Journal of machine learning research*, 11(12), 2010.

[14] Antti Rasmus, Mathias Berglund, Mikko Honkala, Harri Valpola, and Tapani Raiko. Semi-supervised learning with ladder networks. *Advances in neural information processing systems*, 28, 2015.

[15] Richard Zhang, Phillip Isola, and Alexei A Efros. Split-brain autoencoders: Unsupervised learning by cross-channel prediction. In *Proceedings of the IEEE conference on computer vision and pattern recognition*, pages 1058–1067, 2017.

[16] Mark Chen, Alec Radford, Rewon Child, Jeffrey Wu, Heewoo Jun, David Luan, and Ilya Sutskever. Generative pretraining from pixels. In *International conference on machine learning*, pages 1691–1703. PMLR, 2020.

[17] Tejas D Kulkarni, William F Whitney, Pushmeet Kohli, and Josh Tenenbaum. Deep convolutional inverse graphics network. *Advances in neural information processing systems*, 28, 2015.

[18] Tian Ye, Xiaolong Wang, James Davidson, and Abhinav Gupta. Interpretable intuitive physics model. In *Proceedings of the European Conference on Computer Vision (ECCV)*, pages 87–102, 2018.

[19] Xueting Li, Xiaolong Wang, Ming-Hsuan Yang, Alexei Efros, and Sifei Liu. Scraping textures from natural images for synthesis and editing. *ECCV*, 2022.

[20] Daniel Glasner, Shai Bagon, and Michal Irani. Super-resolution from a single image. In *2009 IEEE 12th international conference on computer vision*, pages 349–356. IEEE, 2009.

[21] Yochai Blau, Roey Mechrez, Radu Timofte, Tomer Michaeli, and Lihi Zelnik-Manor. The 2018 pirm challenge on perceptual image super-resolution. In *Proceedings of the European Conference on Computer Vision (ECCV) Workshops*, pages 0–0, 2018.

[22] Sung Cheol Park, Min Kyu Park, and Moon Gi Kang. Super-resolution image reconstruction: a technical overview. *IEEE signal processing magazine*, 20(3):21–36, 2003.

[23] Jianrui Cai, Hui Zeng, Hongwei Yong, Zisheng Cao, and Lei Zhang. Toward real-world single image super-resolution: A new benchmark and a new model. In *Proceedings of the IEEE International Conference on Computer Vision*, 2019.

[24] Radu Timofte, Shuhang Gu, Jiqing Wu, Luc Van Gool, Lei Zhang, Ming-Hsuan Yang, Muhammad Haris, et al. Ntire 2018 challenge on single image super-resolution: Methods and results. In *The IEEE Conference on Computer Vision and Pattern Recognition (CVPR) Workshops*, June 2018.

[25] Priyam Chatterjee and Peyman Milanfar. Is denoising dead? *IEEE Transactions on Image Processing*, 19(4):895–911, 2009.

[26] Javier Portilla, Vasily Strela, Martin J Wainwright, and Eero P Simoncelli. Image denoising using scale mixtures of gaussians in the wavelet domain. *IEEE Transactions on Image processing*, 12(11):1338–1351, 2003.

[27] Antoni Buades, Bartomeu Coll, and Jean-Michel Morel. Non-local means denoising. *Image Processing On Line*, 1:208–212, 2011.

[28] Kostadin Dabov, Alessandro Foi, Vladimir Katkovnik, and Karen Egiazarian. Image denoising by sparse 3-d transform-domain collaborative filtering. *IEEE Transactions on image processing*, 16(8):2080–2095, 2007.

[29] Kostadin Dabov, Alessandro Foi, Vladimir Katkovnik, and Karen Egiazarian. Bm3d image denoising with shape-adaptive principal component analysis. In *SPARS'09-Signal Processing with Adaptive Sparse Structured Representations*, 2009.

[30] Kai Zhang, Wangmeng Zuo, and Lei Zhang. Ffdnet: Toward a fast and flexible solution for cnn-based image denoising. *IEEE Transactions on Image Processing*, 27(9):4608–4622, 2018.

[31] Jingwen Chen, Jiawei Chen, Hongyang Chao, and Ming Yang. Image blind denoising with generative adversarial network based noise modeling. In *Proceedings of the IEEE conference on computer vision and pattern recognition*, pages 3155–3164, 2018.

[32] Jaewon Lee and Kyong Hwan Jin. Local texture estimator for implicit representation function. In *Proceedings of the IEEE/CVF Conference on Computer Vision and Pattern Recognition (CVPR)*, pages 1929–1938, June 2022.

[33] Jingyu Yang, Sheng Shen, Huanjing Yue, and Kun Li. Implicit transformer network for screen content image continuous super-resolution. In M. Ranzato, A. Beygelzimer, Y. Dauphin, P.S. Liang, and J. Wortman Vaughan, editors, *Advances in Neural Information Processing Systems*, volume 34, pages 13304–13315. Curran Associates, Inc., 2021.

[34] Xiaoming Li, Chaofeng Chen, Shangchen Zhou, Xianhui Lin, Wangmeng Zuo, and Lei Zhang. Blind face restoration via deep multi-scale component dictionaries. In *Computer Vision–ECCV 2020: 16th European Conference, Glasgow, UK, August 23–28, 2020, Proceedings, Part IX 16*, pages 399–415. Springer, 2020.

[35] Shi Guo, Zifei Yan, Kai Zhang, Wangmeng Zuo, and Lei Zhang. Toward convolutional blind denoising of real photographs. In *Proceedings of the IEEE/CVF conference on computer vision and pattern recognition*, pages 1712–1722, 2019.

[36] Xintao Wang, Yu Li, Honglun Zhang, and Ying Shan. Towards real-world blind face restoration with generative facial prior. In *Proceedings of the IEEE/CVF Conference on Computer Vision and Pattern Recognition*, pages 9168–9178, 2021.

[37] Zhouxia Wang, Jiawei Zhang, Runjian Chen, Wenping Wang, and Ping Luo. Restoreformer: High-quality blind face restoration from undegraded key-value pairs. In *Proceedings of the IEEE/CVF Conference on Computer Vision and Pattern Recognition*, pages 17512–17521, 2022.

[38] Guangming Liu, Xin Zhou, Jianmin Pang, Feng Yue, Wenfu Liu, and Junchao Wang. Codeformer: A gnn-nested transformer model for binary code similarity detection. *Electronics*, 12(7):1722, 2023.

[39] Vincent Sitzmann, Julien Martel, Alexander Bergman, David Lindell, and Gordon Wetzstein. Implicit neural representations with periodic activation functions. In H. Larochelle, M. Ranzato, R. Hadsell, M.F. Balcan, and H. Lin, editors, *Advances in Neural Information Processing Systems*, volume 33, pages 7462–7473. Curran Associates, Inc., 2020.

[40] Ben Mildenhall, Pratul P. Srinivasan, Matthew Tancik, Jonathan T. Barron, Ravi Ramamoorthi, and Ren Ng. Nerf: Representing scenes as neural radiance fields for view synthesis. In *ECCV*, 2020.

[41] Matthew Tancik, Pratul Srinivasan, Ben Mildenhall, Sara Fridovich-Keil, Nithin Raghavan, Utkarsh Singhal, Ravi Ramamoorthi, Jonathan Barron, and Ren Ng. Fourier features let networks learn high frequency functions in low dimensional domains. *Advances in Neural Information Processing Systems*, 33:7537–7547, 2020.

[42] Nasim Rahaman, Aristide Baratin, Devansh Arpit, Felix Draxler, Min Lin, Fred Hamprecht, Yoshua Bengio, and Aaron Courville. On the spectral bias of neural networks. In *International Conference on Machine Learning*, pages 5301–5310. PMLR, 2019.

[43] Morteza Mardani, Guilin Liu, Aysegul Dundar, Shiqiu Liu, Andrew Tao, and Bryan Catanzaro. Neural ffts for universal texture image synthesis. In H. Larochelle, M. Ranzato, R. Hadsell, M.F. Balcan, and H. Lin, editors, *Advances in Neural Information Processing Systems*, volume 33, pages 14081–14092. Curran Associates, Inc., 2020.

[44] Gaochao Song, Qian Sun, Luo Zhang, Ran Su, Jianfeng Shi, and Ying He. Ope-sr: Orthogonal position encoding for designing a parameter-free upsampling module in arbitrary-scale image super-resolution. In *Proceedings of the IEEE/CVF Conference on Computer Vision and Pattern Recognition (CVPR)*, pages 10009–10020, June 2023.

[45] Ronald Newbold Bracewell and Ronald N Bracewell. *The Fourier transform and its applications*, volume 31999. McGraw-Hill New York, 1986.

[46] Andreas Blattmann, Robin Rombach, Kaan Oktay, and Björn Ommer. Retrieval-augmented diffusion models, 2022.

[47] Ze Liu, Yutong Lin, Yue Cao, Han Hu, Yixuan Wei, Zheng Zhang, Stephen Lin, and Baining Guo. Swin transformer: Hierarchical vision transformer using shifted windows. In *Proceedings of the IEEE/CVF international conference on computer vision*, pages 10012–10022, 2021.

[48] Richard Zhang, Phillip Isola, Alexei A Efros, Eli Shechtman, and Oliver Wang. The unreasonable effectiveness of deep features as a perceptual metric. In *CVPR*, 2018.

[49] Phillip Isola, Jun-Yan Zhu, Tinghui Zhou, and Alexei A Efros. Image-to-image translation with conditional adversarial networks. In *Proceedings of the IEEE conference on computer vision and pattern recognition*, pages 1125–1134, 2017.

[50] Tero Karras, Samuli Laine, and Timo Aila. A style-based generator architecture for generative adversarial networks. In *Proceedings of the IEEE/CVF conference on computer vision and pattern recognition*, pages 4401–4410, 2019.

[51] Ziwei Liu, Ping Luo, Xiaogang Wang, and Xiaoou Tang. Deep learning face attributes in the wild. In *Proceedings of International Conference on Computer Vision (ICCV)*, December 2015.

[52] Tero Karras, Timo Aila, Samuli Laine, and Jaakko Lehtinen. Progressive growing of gans for improved quality, stability, and variation. *arXiv preprint arXiv:1710.10196*, 2017.

[53] Martin Heusel, Hubert Ramsauer, Thomas Unterthiner, Bernhard Nessler, and Sepp Hochreiter. Gans trained by a two time-scale update rule converge to a local nash equilibrium. *Advances in neural information processing systems*, 30, 2017.

[54] Jia Deng, Wei Dong, Richard Socher, Li-Jia Li, Kai Li, and Li Fei-Fei. Imagenet: A large-scale hierarchical image database.

[55] Bee Lim, Sanghyun Son, Heewon Kim, Seungjun Nah, and Kyoung Mu Lee. Enhanced deep residual networks for single image super-resolution. In *The IEEE Conference on Computer Vision and Pattern Recognition (CVPR) Workshops*, July 2017.

[56] Xintao Wang, Liangbin Xie, Chao Dong, and Ying Shan. Real-esrgan: Training real-world blind super-resolution with pure synthetic data. In *International Conference on Computer Vision Workshops (ICCVW)*.

[57] Sachit Menon, Alexandru Damian, Shijia Hu, Nikhil Ravi, and Cynthia Rudin. Pulse: Self-supervised photo upsampling via latent space exploration of generative models. In *Proceedings of the ieee/cvf conference on computer vision and pattern recognition*, pages 2437–2445, 2020.

[58] Minguk Kang, Jun-Yan Zhu, Richard Zhang, Jaesik Park, Eli Shechtman, Sylvain Paris, and Taesung Park. Scaling up gans for text-to-image synthesis. In *Proceedings of the IEEE Conference on Computer Vision and Pattern Recognition (CVPR)*, 2023.

[59] Guan-Horng Liu, Arash Vahdat, De-An Huang, Evangelos A. Theodorou, Weili Nie, and Anima Anandkumar. I$^2$sb: Image-to-image schrödinger bridge, 2023.

[60] Bahjat Kawar, Michael Elad, Stefano Ermon, and Jiaming Song. Denoising diffusion restoration models. In *Advances in Neural Information Processing Systems*, 2022.

[61] Ziwei Luo, Fredrik K. Gustafsson, Zheng Zhao, Jens Sjölund, and Thomas B. Schön. Image restoration with mean-reverting stochastic differential equations. In Andreas Krause, Emma Brunskill, Kyunghyun Cho, Barbara Engelhardt, Sivan Sabato, and Jonathan Scarlett, editors, *Proceedings of the 40th International Conference on Machine Learning*, volume 202 of *Proceedings of Machine Learning Research*, pages 23045–23066. PMLR, 23–29 Jul 2023.

[62] Zongsheng Yue, Jianyi Wang, and Chen Change Loy. Resshift: Efficient diffusion model for image super-resolution by residual shifting, 2023.

[63] Junjie Ke, Qifei Wang, Yilin Wang, Peyman Milanfar, and Feng Yang. Musiq: Multi-scale image quality transformer. In *Proceedings of the IEEE/CVF International Conference on Computer Vision*, pages 5148–5157, 2021.

[64] Google Inc. Open images dataset. https://storage.googleapis.com/openimages/web/index.html, 2016.

[65] Kai Zhang, Yawei Li, Jingyun Liang, Jiezhang Cao, Yulun Zhang, Hao Tang, Radu Timofte, and Luc Van Gool. Practical blind denoising via swin-conv-unet and data synthesis. *arXiv preprint arXiv:2203.13278*, 2022.

[66] Tsung-Yi Lin, Michael Maire, Serge J. Belongie, Lubomir D. Bourdev, Ross B. Girshick, James Hays, Pietro Perona, Deva Ramanan, Piotr Doll'a r, and C. Lawrence Zitnick. Microsoft COCO: common objects in context. *CoRR*, abs/1405.0312, 2014.

# Appendix Table of Contents

## Appendix Outlines

The Appendix is structured as follows: We present the broader impact of our work, in Sec. A. We provide insights for the theories presented in the main paper, we discuss more on (i) frequency analyses at Sec. B.1; (ii) if the frequencies need to be regularized during training, at Sec. B.2, and (iii) the limitations, at Sec. B.3. We present additional implementation details and experimental results, in Sec. C and D. Most qualitative results are shown in the last few pages.

## A    Broader Impact

With the presented applications of face and natural image restoration, the societal impact of the CosAE can be far-reaching. Accurate image restoration is crucial in various domains, including forensic analysis, medical imaging, and digital art preservation. CosAE's ability to restore high-quality images while preserving fine-grained details has the potential to enhance visual content in these fields. By enabling the restoration of images with unknown degradation types, CosAE offers a valuable tool for image analysis and enhancement, leading to improved decision-making, enhanced diagnostics, and enhanced cultural preservation. Furthermore, the generalizability of CosAE can contribute to advances in computer vision applications, such as image recognition, object detection, and scene understanding, with broader implications in fields like autonomous vehicles, robotics, and augmented reality.

On the potential downsides, one key concern is the bias in the training data used to develop the model. If the training data is not diverse and representative of different demographics, there is a risk of perpetuating biases or inequalities in the restored images, particularly in the case of face restoration. This can lead to unintended consequences, such as reinforcing societal stereotypes or discrimination. Therefore, we choose to use the training sets on the face-related tasks, e.g., FFHQ and CelebA, that are inclusive, diverse, and carefully curated to mitigate biases and promote fairness in image restoration applications. Additionally, transparency and ethical considerations should be prioritized to address concerns related to privacy, consent, and the responsible use of restored images, particularly in sensitive domains such as healthcare or criminal investigations. However, it is important to note that the primary objective of CosAE is pixel-level restoration rather than recognition or synthesis of faces. This distinction implies that there may be relatively fewer concerns associated with the use of CosAE compared to other models in the face domain, such as face recognition or synthesis models.

## B    Discussions

While major designs are presented in the main paper, here, we provide more insights and limitations of CosAE.

### B.1    Frequency Analyses

We visualize the learned frequencies $u, v \in \mathcal{R}^{c \times 2}, c = 256$ in Figure. 7. While initialized uniformly between $[0, T/2]$ (see Sec.3.4), the learned frequencies effectively capture both low- and high-frequencies without significant drift from their initial values. We note that unlike [32], where each image has its individual set of frequencies, which can be compared with its ground truth frequency values computed via FFT, in our case, all images will share the same set of frequencies. Along with the design where the frequencies are from the latent space, as such, there is no such "ground truth" we can compare with. However, our design is closer to the classic discrete Fourier transform, where frequencies are often uniformly sampled and are not input-conditioned.

**Initializing with low frequencies.**   To further quantitatively study the potential effects for frequency, we conduct another ablation study: we initialize $u, v$ with low frequencies only, i.e., $u, v \in (0, 0.5]$, denoted as *-LF*, and train the model with L1 loss only to exclude any factors (e.g., the discriminator or the LPIPS loss) that may generate high-frequency details. As demonstrated in Table. 6, without high-frequency initialization, as proposed in CosAE, *CosAE-LF* shows degraded performance on all the matrices.

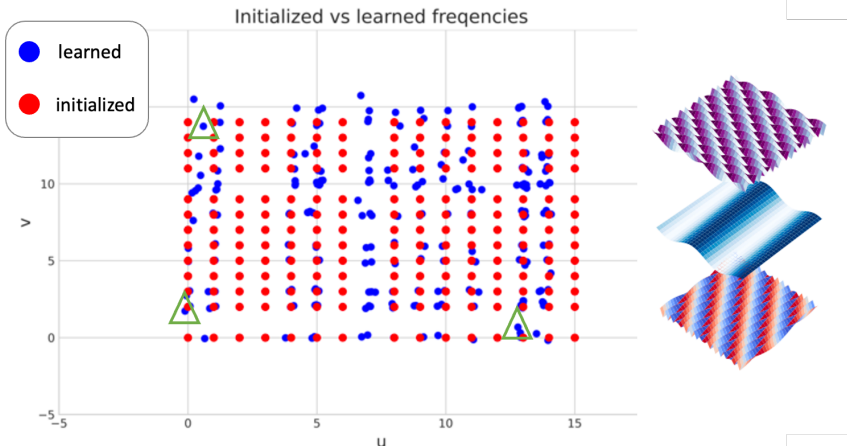

*Figure 7.* **Initialized frequencies vs. learned frequencies.** While initialized uniformly between $[0, T/2]$ (see Sec. 3.4), the learned frequencies effectively capture both low- and high-frequencies without significant drift from their initial values. 3 pair of learned frequencies $u, v$, marked by green $\triangle$ among the points, are visualized as 2D waves on the right.

*Table 5.* **FR-SR evaluation for uniform vs learnable** $(u, v)$. We denote the ablation (suggested by **Reviewer Gqnq**) for using freezed, uniformly sampled frequencies as CosAE-uniform.

| | FID-15k↓ | | | | LPIPS↓ | | | | PSNR (db)↑ | | | | SSIM↑ | | | |
|---|---|---|---|---|---|---|---|---|---|---|---|---|---|---|---|---|
| LR-res | 32 | 48 | 64 | 128 | 32 | 48 | 64 | 128 | 32 | 48 | 64 | 128 | 32 | 48 | 64 | 128 |
| CosAE-uniform (G) | 13.09 | 10.80 | 10.28 | 10.49 | 0.28 | 0.25 | 0.23 | 0.22 | 22.69 | 23.51 | 24.05 | 24.21 | 0.63 | 0.65 | 0.67 | 0.68 |
| **CosAE (G)** | **12.81** | **8.67** | **8.12** | **7.86** | **0.24** | **0.19** | **0.17** | **0.14** | **23.65** | **25.37** | **26.66** | **27.51** | **0.67** | **0.72** | **0.76** | **0.78** |

**Does non-learnable frequencies lead to performance degradation?** We found that fixing frequencies $(u, v)$ to a uniform grid caused a slight performance drop, reinforcing the need for learnable frequencies for a more generalizable model. We retrained the model with fixed uniform sampling, finding that CosAE-uniform underperformed CosAE across metrics, as shown in Table 5. Uniform sampling, while initially dense with $T = 32$, leads to sparse frequency coverage as basis maps or resolution increase. This sampling restricts adaptability and imposes a strict grid format, limiting modeling accuracy. Thus, learnable frequencies enhance both performance and design flexibility.

## B.2 Frequency Regularization During Training?

The flexible output resolution training strategy, as introduced in Sec. 4.1 for training Flexible-resolution Super-resolution (FR-SR), yields a potential issue: When adjusting $T \in (0, T_{max}]$, frequencies of any $u$ and $v$ that are greater than $T/2$ will result in aliased 2D waveform maps. Will this issue degrade performance? Or in other words, is non-aliased Cosine basis functions necessary for CosAE? **The answer is NO**, as we detailed the validation experiments in the following.

**Training with Non-aliased Basis Functions.** To investigate whether maintaining non-aliased basis functions the necessity for CosAE, we design and compare with a variant, namely *-NoAls*. Specifically, we introduce a channel-dropout approach to adaptively obtain valid frequencies for dynamically varying output resolution. Given $T \in (0, T_{\max}]$, we mask out any basis functions $\mathcal{H}_k \in \{\mathcal{H}\}_{T \times T \times c}$ when their $\min(u_k, v_k) \geq T/2$. Consequently, when reconstructing a smaller-sized output, only a subset of the basis functions and bottleneck features are utilized. Intuitively, the design also indicates that a smaller image requires fewer bits in the bottleneck (see Figure. 8).

As detailed in Table. 6, *-NoAls* even outperforms the reported CosAE, which is trained without removing the aliased basis functions, in terms of FID scores. On the other hand, it still underperforms in terms of LPIPS, PSNR and SSIM, indicating less consistent results compared to the ground-truth images. Thus, we conclude that: (a) training CosAE without maintaining a non-aliased set of Cosine basis functions achieves satisfactory results and does not degrade the performance, as we reported

in the main paper; (b) the approach *-NoAls* is also a valid alternation that supports even more compressive bottleneck representation when smaller output resolution is desired.

## B.3 Limitations

**Out-of-distribution Generalization.** In contrast to implicit image representation methods for continuous super-resolution [10, 32, 33], where out-of-distribution up-sampling can be achieved by interpolating the coordinate input maps, it is not advisable for CosAE to follow a similar approach by directly increasing the value of $T$ during inference. This is because frequencies $u$ and $v$ that are greater than $T_{max}/2$, which is fixed during training, have not been learned and thus crucial high-frequency details may be lost.

Alternatively, CosAE can still generalize super-resolution to higher ratios, e.g., super-resolve face images that are equal to or greater than $32 \times 32$ to $512 \times 512$, simply by resizing the input image to $512 \times 512$. However, the face model's generalization capability is limited since the model has not been exposed to high-resolution face patches during training. This is in contrast to the natural image domain model, which is trained on random crops of diverse natural images, allowing for a broader range of generalization. Thus, we suggest two solutions: First, one can utilize face images with a variety of resolutions and crop patches of size $256 \times 256$. It is important to include higher resolution patches to capture the desired level of detail in the training process. Secondly, if a specific resolution, such as $512 \times 512$, is desired, one can further fine-tune the FR-SR face model on the specific resolution (all network architectures and training settings are kept the same). The model typically converges super fast, e.g., within 100 iterations, during this fine-tuning process. We show the qualitative performance of a $512 \times 512$ face model, which is obtained by fine-tuning the FR-SR face model (the second approach), in Figure. 11.

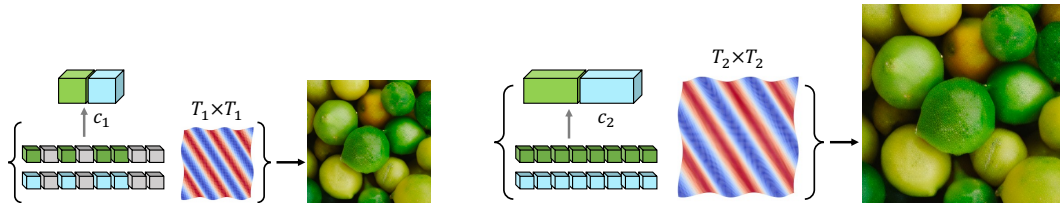

*Figure 8.* **Training with Non-aliased Basis Functions.** Left: when $T$ is small, the corresponding high-frequency coefficients stored in the bottleneck are masked, resulting in a smaller output image; Right: when $T = T_{max}$, all the bottleneck features are adopted to have a full-sized output image.

*Table 6.* **Additional FR-SR evaluation on the val splits of FFHQ+CelebA faces (15k images)**. We compared our CosAE with (a) the variant that initialized with only low-frequency set of $u, v$, as *CosAE-LF* (Sec. B.1); and (b) the variant *-NoAls*, that maintain non-aliased basis function only during training (Sec. B.2).

| | FID-15k↓ | | | | LPIPS↓ | | | | PSNR (db)↑ | | | | SSIM↑ | | | |
|---|---|---|---|---|---|---|---|---|---|---|---|---|---|---|---|---|
| LR-res | 32 | 48 | 64 | 128 | 32 | 48 | 64 | 128 | 32 | 48 | 64 | 128 | 32 | 48 | 64 | 128 |
| CosAE-LF | | N/A | | | 0.38 | 0.33 | 0.31 | 0.30 | 23.99 | 25.88 | 26.99 | 27.80 | 0.68 | 0.74 | 0.76 | 0.78 |
| **CosAE** | | N/A | | | **0.36** | **0.32** | **0.28** | **0.26** | **24.34** | **26.18** | **27.52** | **28.51** | **0.70** | **0.75** | **0.79** | **0.81** |
| CosAE-NoAls (G) | 10.78 | 8.12 | 7.99 | 7.53 | 0.25 | 0.29 | 0.18 | 0.17 | 23.55 | 24.96 | 25.88 | 26.30 | 0.66 | 0.70 | 0.73 | 0.74 |
| **CosAE (G)** | 12.81 | 8.67 | 8.12 | 7.86 | **0.24** | **0.19** | **0.17** | **0.14** | **23.65** | **25.37** | **26.66** | **27.51** | **0.67** | **0.72** | **0.76** | **0.78** |

**Common Artifacts.** When restoring extremely low-resolution or degraded images, CosAE also presents artifacts. See the tree leaves in the second row of Figure. 10 and the fur textures in the last row of Figure. 10. Interestingly, the artifacts are similar to 2D sinusoidal waves – which is very different from the conventional CNN/transformer-based models that commonly produce over-smooth or aliased regions, see Figure. 10, the first and the second columns.

## C Implementation Details

We further introduce the configuration of the CosAE models for both the face and the natural image domains, with a particular emphasis on the encoder and decoder given sufficient description is provided for the basis construction module in the main paper. We include key training details to ensure code reproduction. Codes will be released upon legal approval.

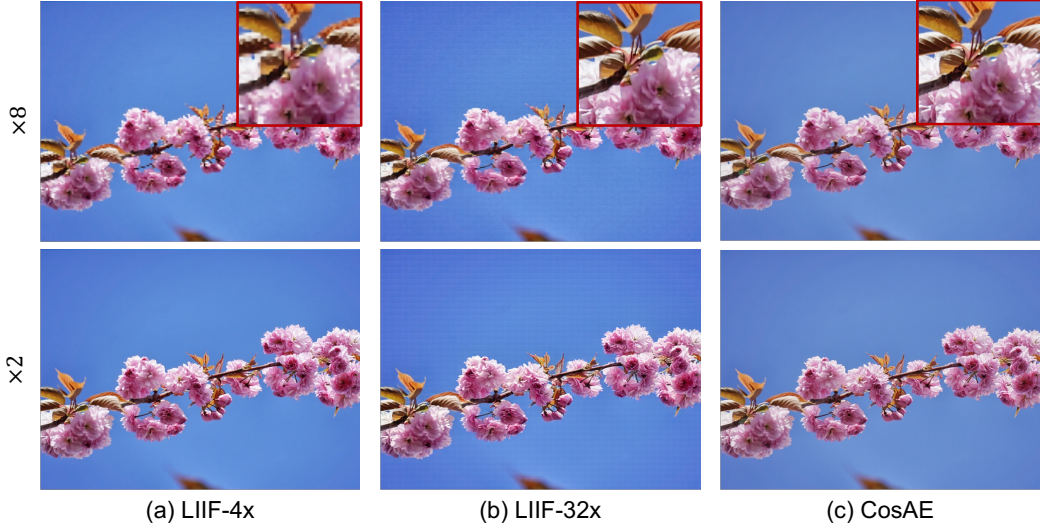

| (a) LIIF-4x | (b) LIIF-32x | (c) CosAE |

*Figure 9.* **Wide vs. narrow bottlenecks for DIV2k.** By comparing upsampling with $8\times$ and $2\times$, (b) LIIF-32x and (c) CosAE that have narrow bottlenecks presents much more consistent super-resolution results, compared with that of (a) LIIF-4x, with a wide bottleneck.

**Network Architectures.** We follow [6] to design the encoder E and the decoder D module, where the design logic and the detailed descriptions can be found from its appendix. Based on the stride, for the face model, we configure 7 down-sampling blocks resulting in $64\times$ down-sampling in the bottleneck; Likewise for natural images, we configure 6 down-sampling blocks to accommodate its $32\times$ down-sampling stride. We configure 2 ResNet blocks at each resolution, and 2 vanilla attention blocks at the last two down-sampling blocks, for both models. We adopt the same decoder for face and natural image models, which retains only 1 upsampling layer, and is configured with 7 ResNet blocks in total.

Specifically in the decoder, in addition to the aforementioned blocks, we also introduce two attention layers besides the first two ResNet blocks. We restrict the attention layers to nonoverlapping local regions – *only within each cosine basis function*, as shown in the right red block of $\{\mathcal{H}\}$ in Figure. 1 in the main paper. This design (i) allows inference of arbitrary resolution input images, e.g., super-resolution of the DIV2K validation set, as shown in Figure. 10 and Figure. 5 in the main paper; and (ii) significantly reduces memory consumption, especially for high-resolution images.

**Training Details.** For all models, we apply the weight $\lambda$ for the discriminative loss of the patch $\mathcal{L}_{GAN}$ as 0.8, while fixing the weight for the regression loss $\mathcal{L}_{rec}$ to 1 (see Sec. 3.6 in the main paper). We adopt the same configuration for the Adam optimizer as introduced in the VQVAE training pipeline in [6, 7], for all the models. We train all the models up to convergence. To reiterate the details described in the main paper: (i) for FR-SR task (Sec.4.1) in both face and natural image domains, we randomly sample $T < T_{max}$ to allow for flexible resolution output. (ii) we fix the $T$ to $T_{\max}$ for both image restoration tasks (Sec 4.3) and the fixed-ration super-resolution for ImageNet (Sec 4.4). (iii) Except for face image restoration tasks where models are trained with $512 \times 512$ input images to align with the previous work, all the other models are trained using $256 \times 256$ images or patches.

## D  Additional Experimental Results

### D.1  Arbitrary-Resolution Super-resolution

**Inference with Varying $T$.** In the main paper, we evaluate flexible-resolution super-resolution by re-scaling the input images to the desired output resolution and then performing Autoencoder reconstruction. However, in this section, we present an alternative approach for achieving arbitrary-resolution super-resolution. Similar to [10, 32, 33] where the output resolution is controllable by configuring the size of the coordinate map, in CosAE, the output resolution can be flexibly changed

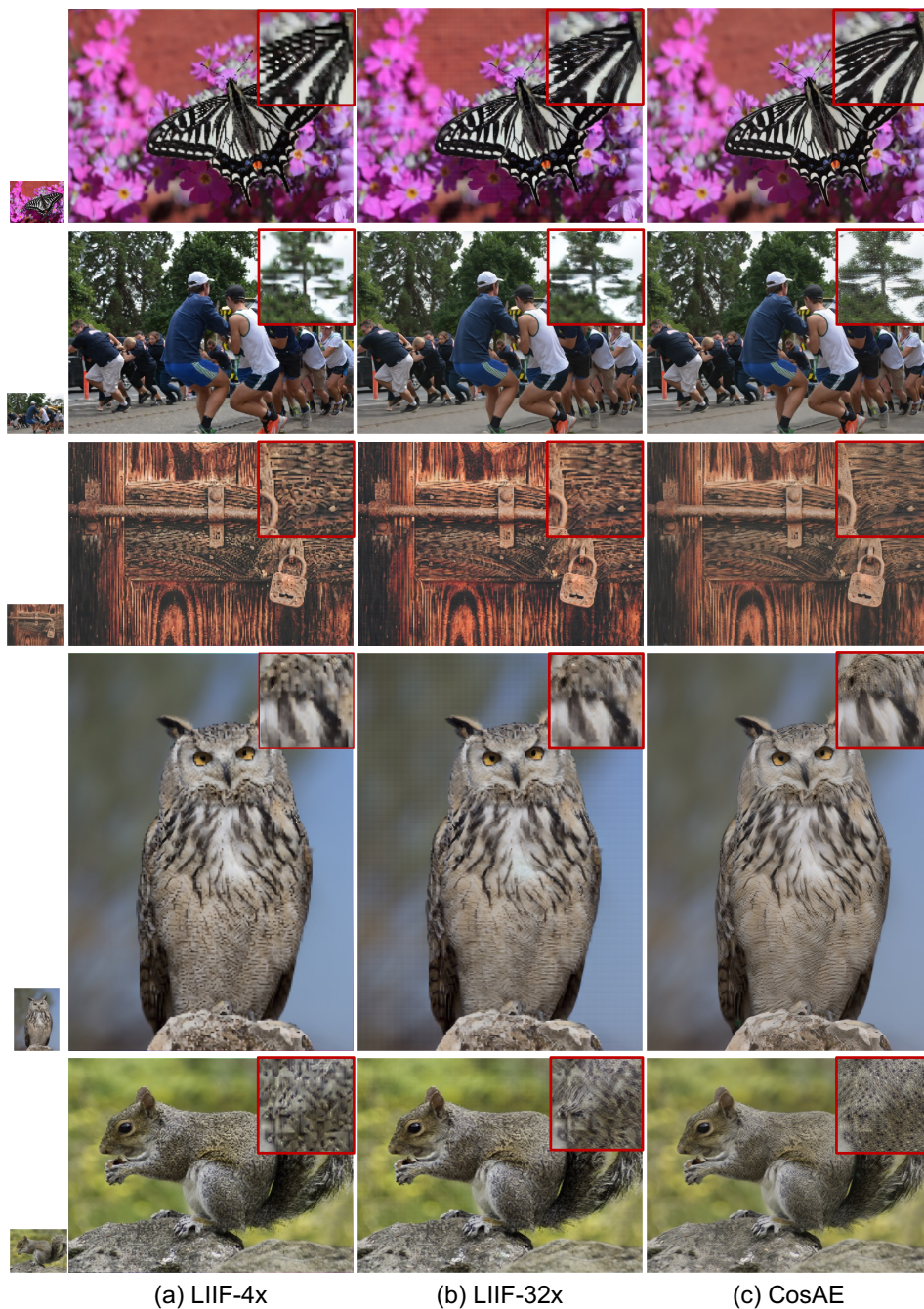

|  |  |  |
|---|---|---|
| (a) LIIF-4x | (b) LIIF-32x | (c) CosAE |

*Figure 10.* **More arbitrary-resolution super-resolution on DIV2K [24]**. We showcase $8\times$ results (the LR input images are illustrated in the left-most column accordingly) as in the Figure. 5 in the main paper. Zoom in to see more details, especially within close-ups.

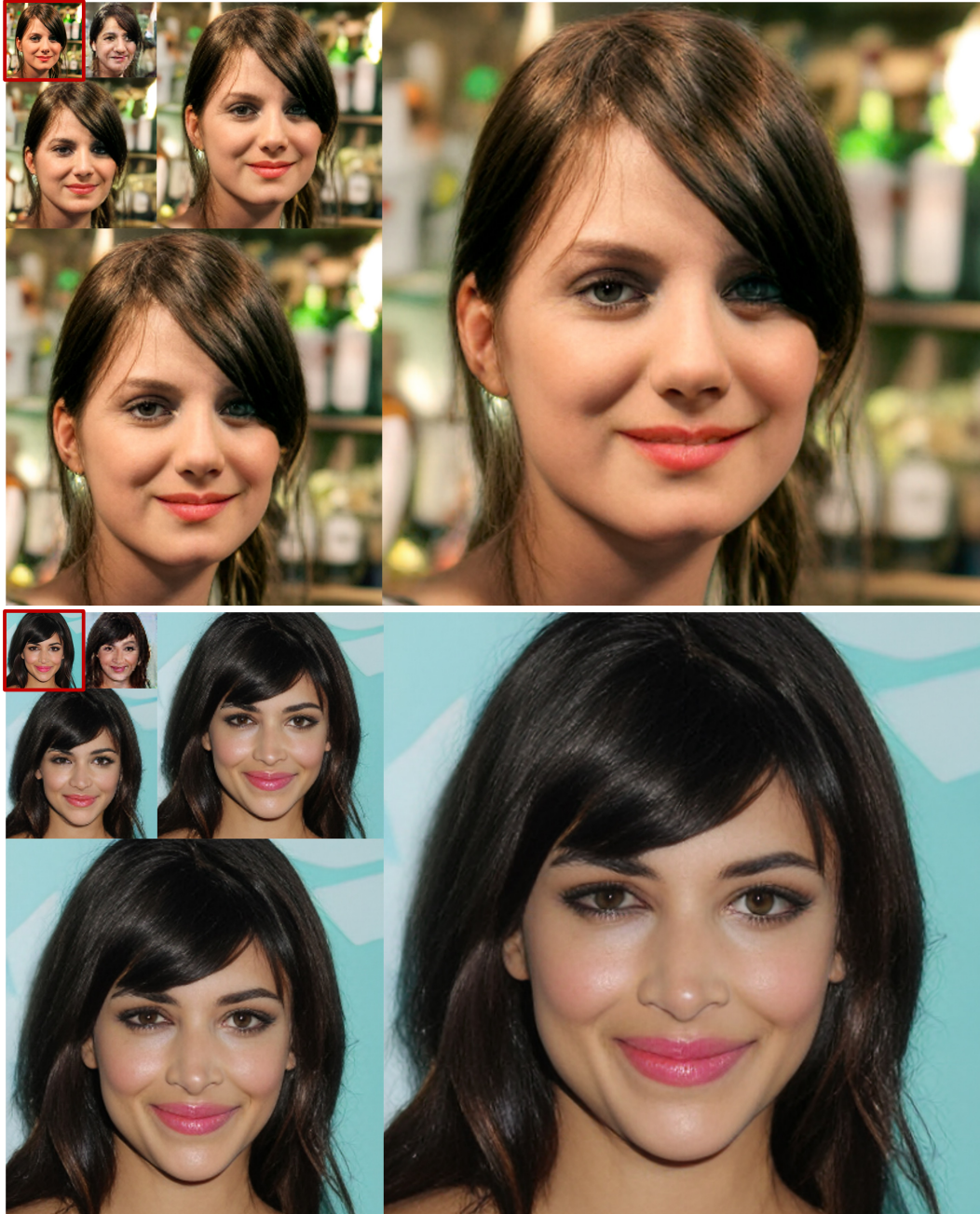

*Figure 11.* **Super-resolving a** $64 \times 64$ **face image by 1 to 8 times, via varying the** $T$. We perform arbitrary-resolution super-resolution in a different way, by varying $T \in [T_{\min}, T_{\max}]$ that is analogous to interpolating the coordinate maps in [10, 32, 33]. The images in the red bounding boxes denote the LR inputs. HR images with the size of $[64, 128, 192, 320, 512]$ are shown.

by varying $T < T_{\max}$. As aforementioned, we show in Figure. 11 that to super-resolve an $64 \times 64$ image by 1 to 8 times, where the model is fine-tuned on the face FR-SR model on $512 \times 512$ training images from the training splits of FFHQ [50] and CelebA-HQ [51] (which is consistent with training the face FR-SR model), to present higher resolution reconstruction results.

**More Results for Face Image Super-resolution.** We show more qualitative evaluations with comparisons to the CosAE variants introduced at Sec.4.2, i.e., the *nocos*, *imcos* and *FT*, in Figure. 12, in the case of up-sampling the $32 \times 32$ LR input images (Figure. 12 (a)) to $256 \times 256$. We do not include results of the LIIF-64x and the LIIF-4x models here since they are significantly worse, as

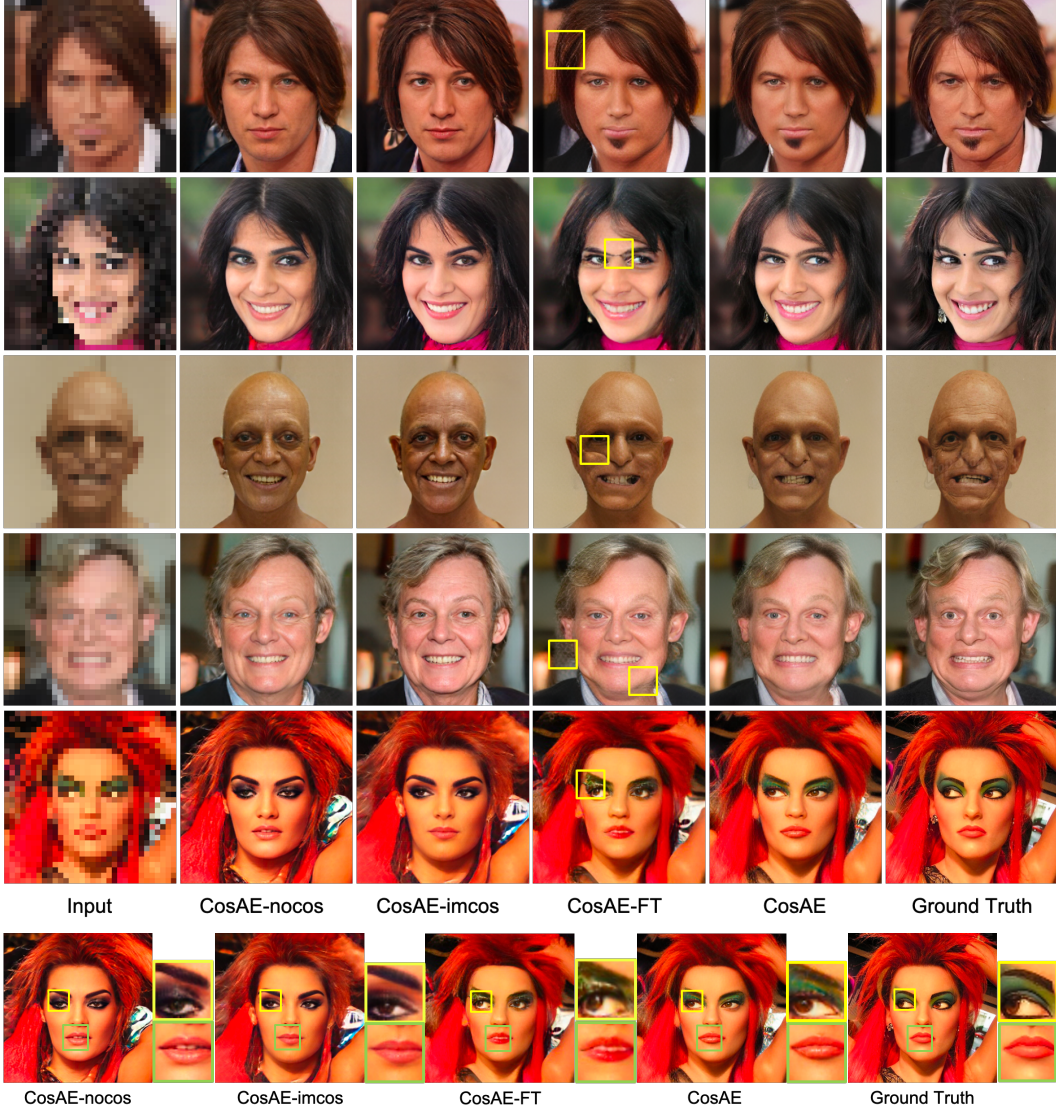

*Figure 12.* **Qualitative evaluation of Face Image Super-resolution**. We compare CosAE with its variants, the *nocos*, *imcos* and *FT*, with $x8$ super-resolution. While the *nocos* and *imcos* are likely to change the identity and style of the original subjects, the *FT* yields artifacts in the hair and facial regions, as shown in the close-ups. Zoom-in to see details.

already demostrated in Figure.3 in the paper. We demonstrate that CosAE significantly outperforms the others. Specifically, both *nocos* and *imcos* are prone to completely change the facial style (row 1 and 5), expressions (row 4), identities (row 1,3,4), although their image quality are visually present. The *FT* presents degraded details on the texture regions, such as hair and facial lines. We show zoomed regions of one example in the last row to contrast their differences.

**More Results for Natural Image FR-SR.** We showcase the $8\times$ super-resolution results on the validation split of the DIV2k [24] dataset. Please note that all models are designed to upsample low-resolution images by a factor ranging from 1 to 8, as detailed in Sec. 4.1 of the main paper.

In Figure. 9, we demonstrate the importance of a narrow bottleneck in the natural image domain. Considering that the base and super-resolution models for natural images possess a relatively wider bottleneck compared to face models – specifically with a down-sampling stride of 32x – the consistency across varying super-resolution ratios is less conspicuous, especially as evidenced in Table 2 of the main paper. Nonetheless, our observations reveal that noticeable artifacts are more pronounced in the results generated by models with wider bottlenecks, for instance, LIIF-4x as displayed in Figure.

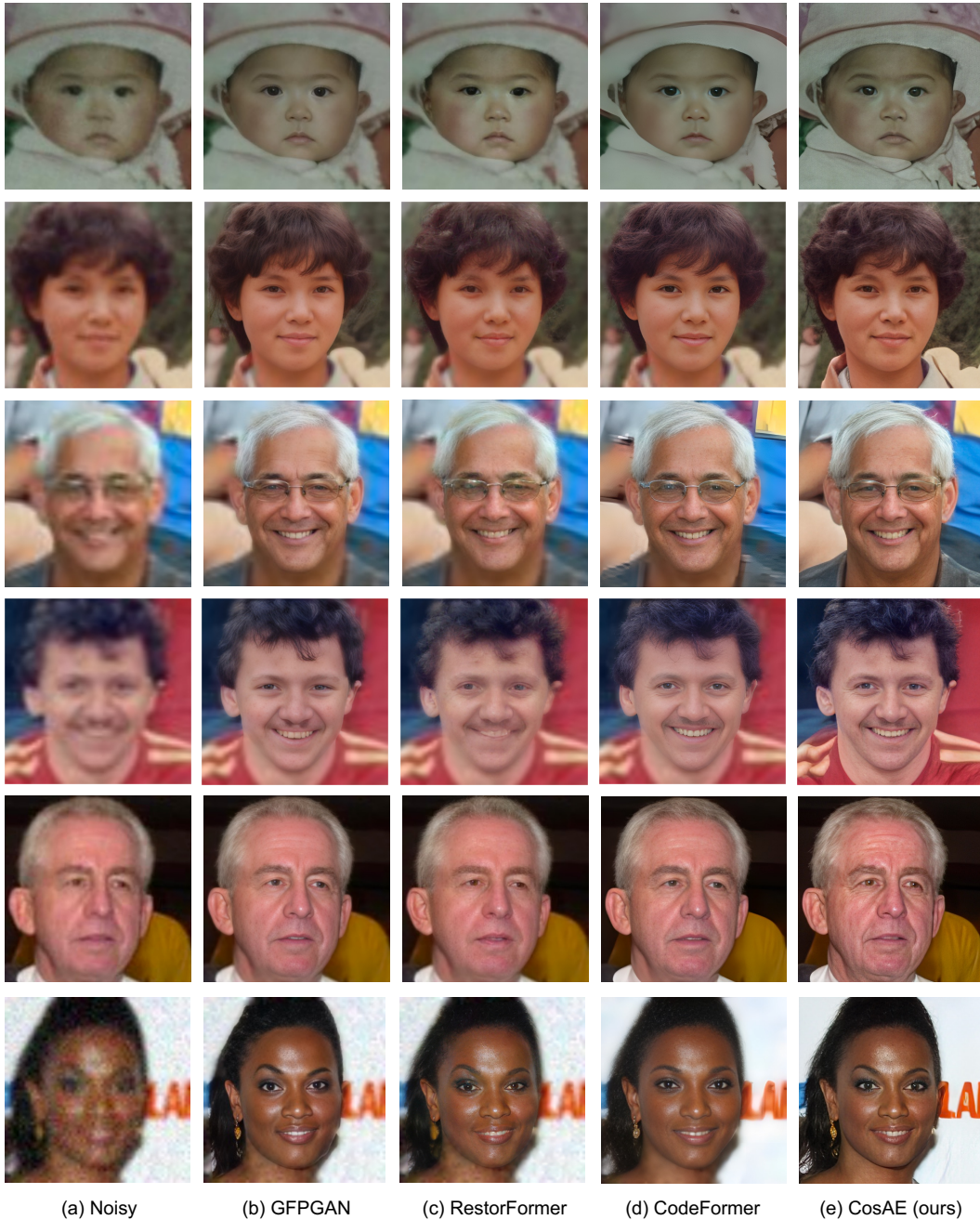

(a) Noisy      (b) GFPGAN      (c) RestorFormer      (d) CodeFormer      (e) CosAE (ours)

*Figure 13.* Comparisons with the state-of-the-art **face restoration approaches on degraded images**. We show results on both the synthetic degraded image dataset and the real-world degraded image datasets. That is, row 1-2 for the WebPhoto-Test dataset (median degradation), row 3-4 for the WIDER-Test [38] dataset (server degradation), row 5 for the LFW-Test dataset (light degradation), and row 6 for the CelebA-Test synthetic dataset. Note that we enable *Face Alignment* and *Background Enhancement* for CodeFormer to obtain the best performance, and test the latest version v1.4 for GFPGAN [36].

*Table 7.* **Quantitative evaluation on the real-world face degraded images.** Among them, LFW-Test, WebPhoto-Test, and WIDER-Test include samples exhibiting a range of degradation levels, from light to severe.

| Model & Metrics | LFW-Test | | WebPhoto-Test | | WIDER-Test | |
|---|---|---|---|---|---|---|
| | FID | MUSIQ | FID | MUSIQ | FID | MUSIQ |
| DFDNet | 62.57 | 67.95 | 100.68 | 63.81 | 57.84 | 59.34 |
| PULSE | 64.86 | 66.98 | 86.45 | 66.57 | 73.59 | 65.36 |
| GFPGAN | 49.96 | 68.95 | 87.35 | 68.04 | 40.59 | 68.26 |
| CodeFormer | 52.02 | 71.43 | 78.87 | 70.51 | 39.06 | 69.31 |
| CosAE | 60.78 | **71.75** | **76.16** | **72.10** | 44.64 | **72.62** |

*Table 8.* **Quantitative evaluation with image restoration on COCO-Test with synthetic corruption.** The FID scores are computed on the COCO validation split, which contains 5000 images.

| Model | FID-5k↓ | LPIPS↓ | PSNR↑ | SSIM↑ |
|---|---|---|---|---|
| SCUNet | 45.47 | 0.41 | 22.42 | 0.62 |
| CosAE | **22.34** | **0.31** | **22.85** | **0.63** |

9 (a), compared to the other two models with a narrow bottleneck. Similar trends can be observed in Figure. 10. Furthermore, even more substantial discrepancies are apparent when comparing its 8x super-resolution results with those of 2x.

It worth noting that in the re-implemented LIIF [10], we use nearest-neighbor instead of weighted interpolation (denoted as local ensemble in the original paper) on top of the bottleneck. This is because the weighted interpolation doesn't ensure convergence when coupled with the new encoder and decoder networks. As a result, the outputted images exhibit grid-like artifacts (see Figure. 9 and 10 (b)), especially those with large super-resolution ratios.

## D.2 Face Image Restoration

We provide more quantitative evaluation on the real-world datasets, including LFW-Test, WebPhoto-Test, and WIDER-Test, in Table. 7. It is important to acknowledge that the FID score may not provide an accurate assessment of realism when dealing with a relatively small test set, such as in the aforementioned datasets. In line with the approach in [38], we additionally incorporate MUSIQ [63] into our analysis.

Furthermore, we show more qualitative results in Figure. 13, compared with the SOTA approaches. For both real-world (row 1-5) and synthetic (row 6) degraded images, CosAE tends to reconstruct more textures in the hair and skin regions. For example, the model tends to synthesize more wrinkles, freckles, beards, etc., on the face, than to generate a smoother one, see Figure. 13, row 3-5. We enable *Face Alignment* and *Background Enhancement* for CodeFormer to obtain the best performance, and test the latest version v1.4 for GFPGAN [36].

## D.3 4× Super-Resolution on ImageNet

We follow the setting in [7] to fine-tune a $4\times$ super-resolution model on top of the natural image FR-SR models pretrained on ImageNet. That is, image degradation models are fixed to a bicubic interpolation, with $4\times$ down-sampling to the $256 \times 256$ images from the training split. In Figure 14, we conducted a comparison with the Latent Diffusion Model (LDM) [7][1]. Importantly, we benchmarked our method against the qualitative evaluations outlined in [7] and [58]. These were trained on the same dataset, as detailed in Table 4 of the main paper. It's worth mentioning that the model referenced in Table 4 of the main paper may not align with the released model utilized in Figure 14, which was pretrained on OpenImages [64].

## D.4 Natural Image Restoration

We compare our natural image restoration model with SCUNet [65], a model that shares similar configurations to ours. Notably, neither approach utilizes ground-truth noisy-clean pairs while training. Similar to the face restoration model, we fine-tune the FR-SR model from the natural image

domain with synthetically degraded images. To do so, we follow the design of image corruption operators in SCUNet [65] to produce training pairs online on the ImageNet training split. To evaluate the approach, we further introduce a synthetic dataset produced on the COCO validation split [66], namely COCO-Test, using the same corruption operators at training. Adopting the metrics as Table. 3 in the main paper (except IDD, which is not applicable for images other than faces), the quantitative evaluation on the synthetic dataset is shown in Table. 8.

We further showcase the image restoration results on both COCO-Test and a real-world noisy image dataset RNI15 [65], in Figure. 20. On a broad level, CosAE and SCUNet [65] exhibit different propensities. CosAE typically emphasizes the preservation of texture and intricate structural details, while SCUNet shows a tendency to maintain the integrity of larger structures.

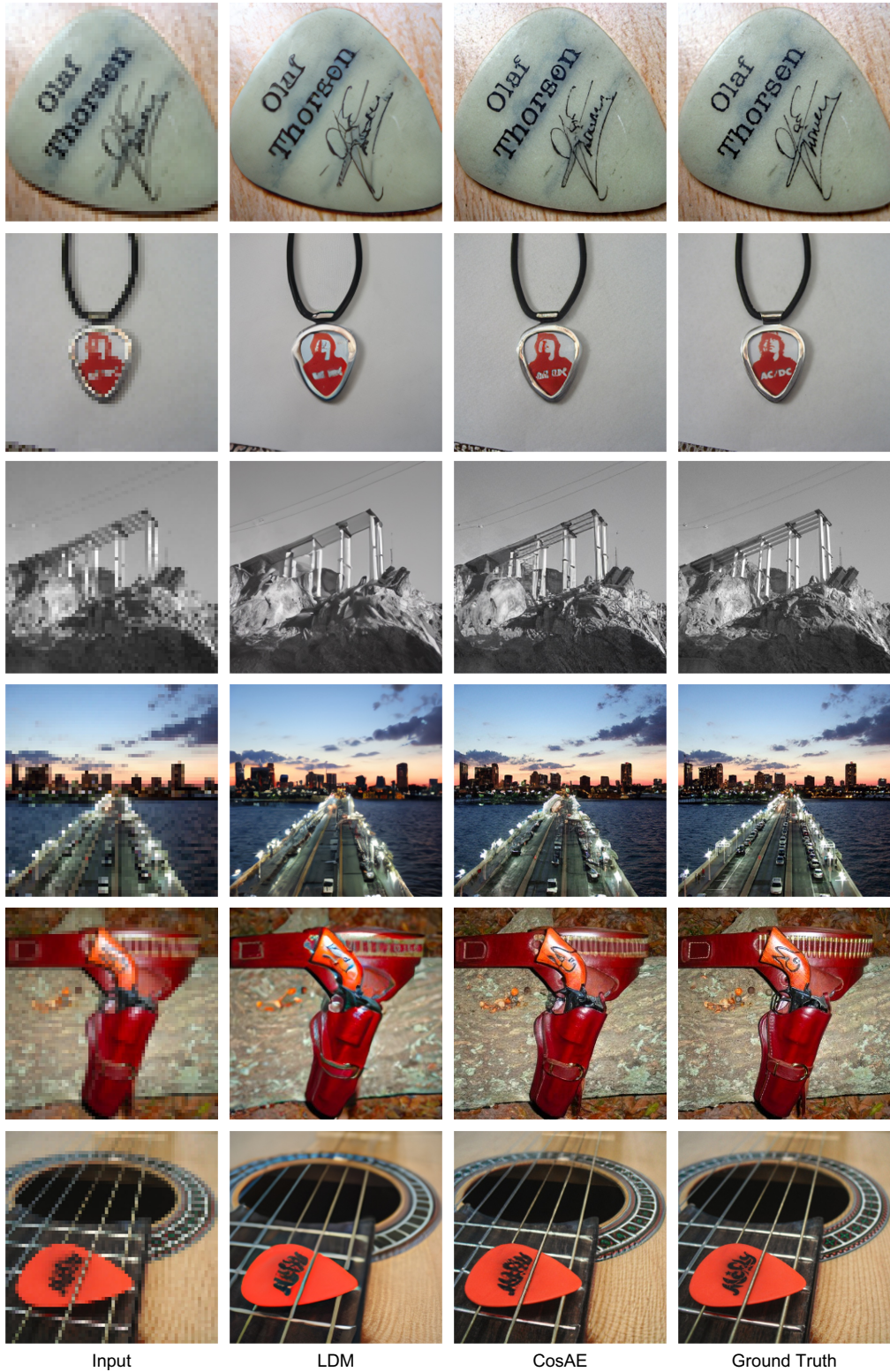

| Input | LDM | CosAE | Ground Truth |

*Figure 14.* Comparisons with LDM on $4\times$ **super-resolution on ImageNet**. In comparison, the CosAE produces more fine-grained details, in terms of both the structure and the texture. Zoom in to see details.

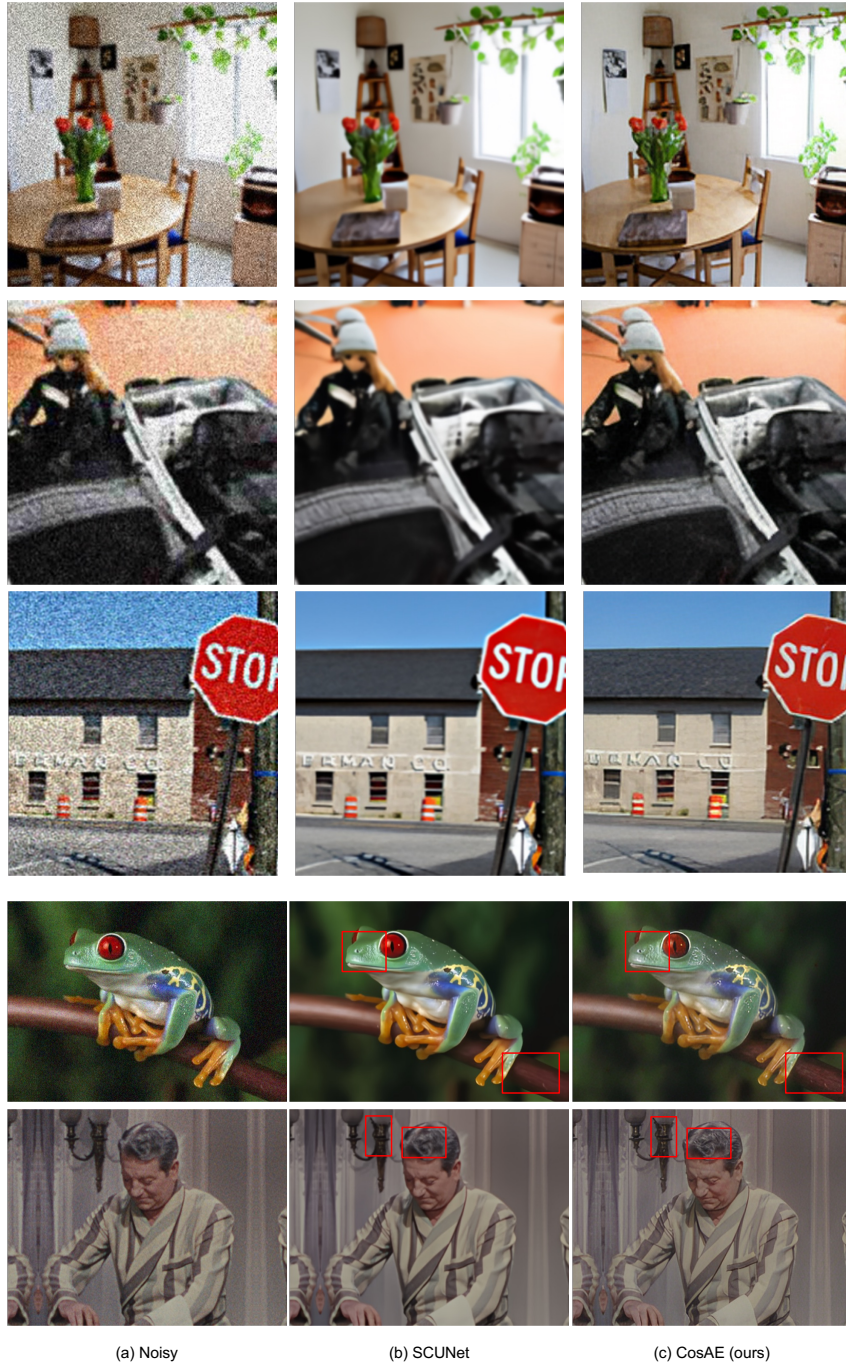

| (a) Noisy | (b) SCUNet | (c) CosAE (ours) |

*Figure 15.* **Qualitative evaluation for image restoration.** We compared CosAE with SCUNet [65]. We show denoising results on the synthetic noisy dataset created from the validation split of COCO [66], in the first 3 rows. We also show the denoising results on **real-world** noisy images from RNI15 [65], in the last 2 rows. Zoom in to see details.

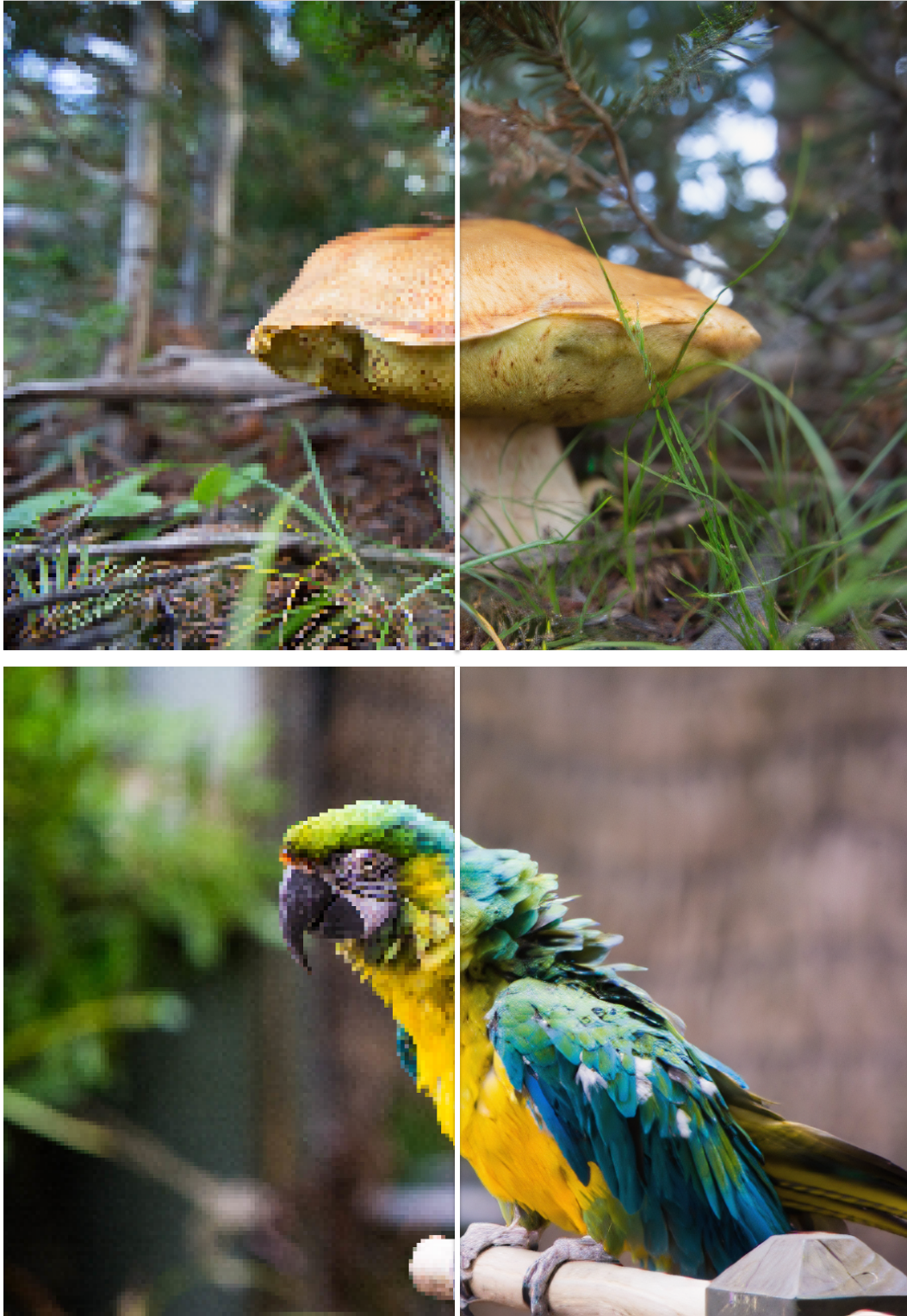

*Figure 16.* ×8 Super-Resolution for Natural Images.

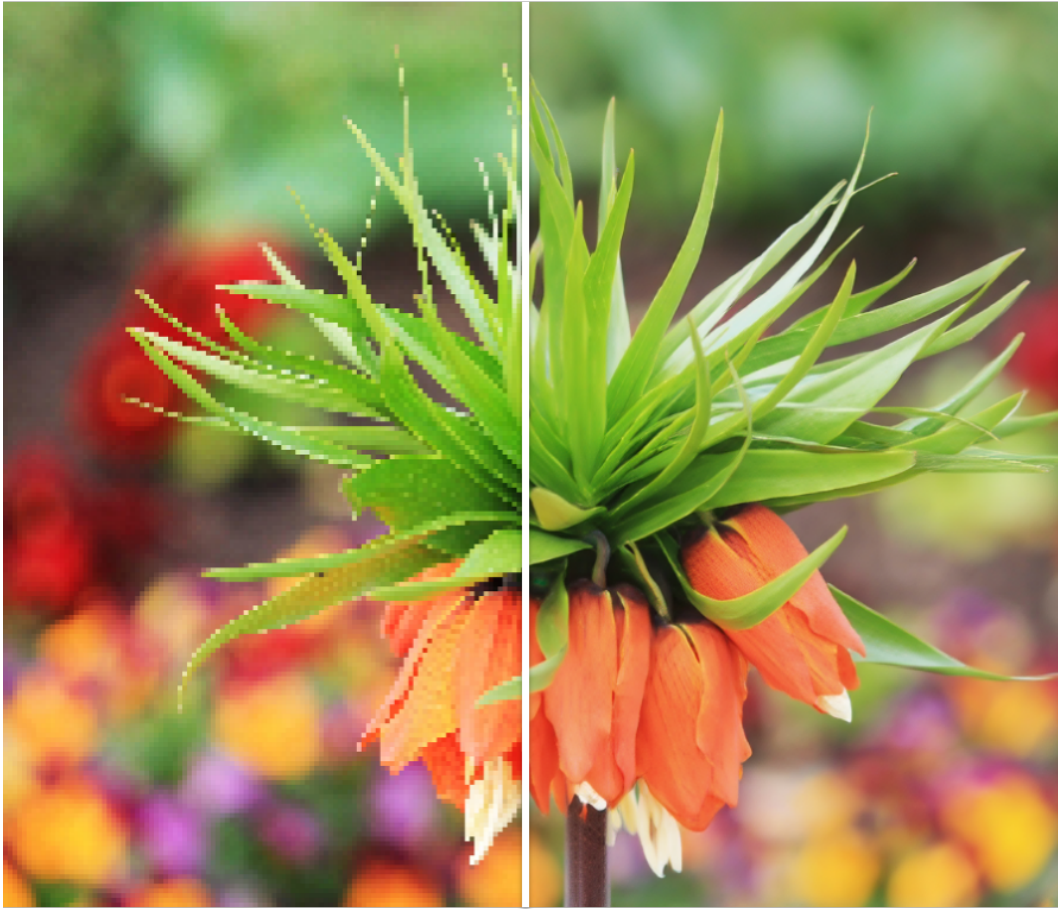

*Figure 17.* ×8 Super-Resolution for Natural Images.

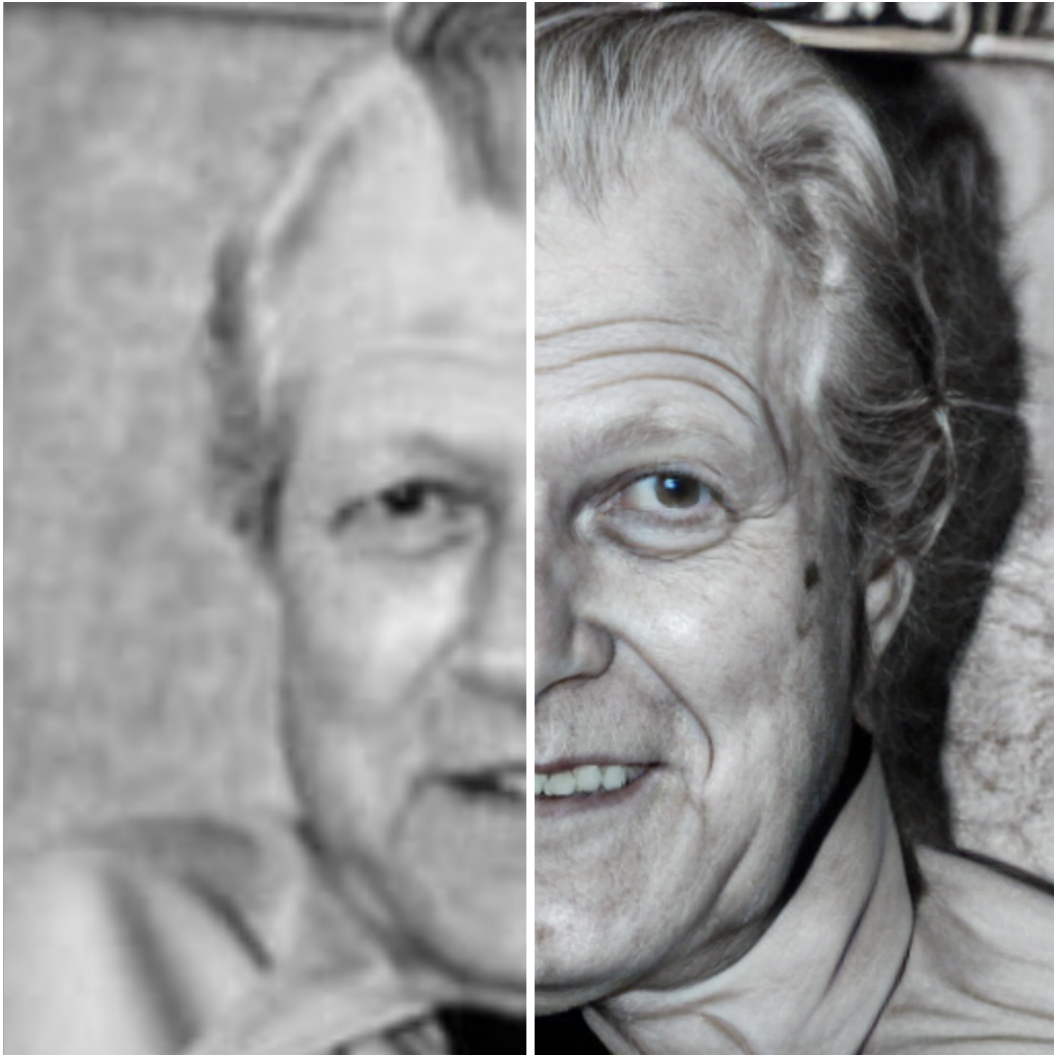

*Figure 18.* Blind face image restoration, with $512 \times 512$ resolution.

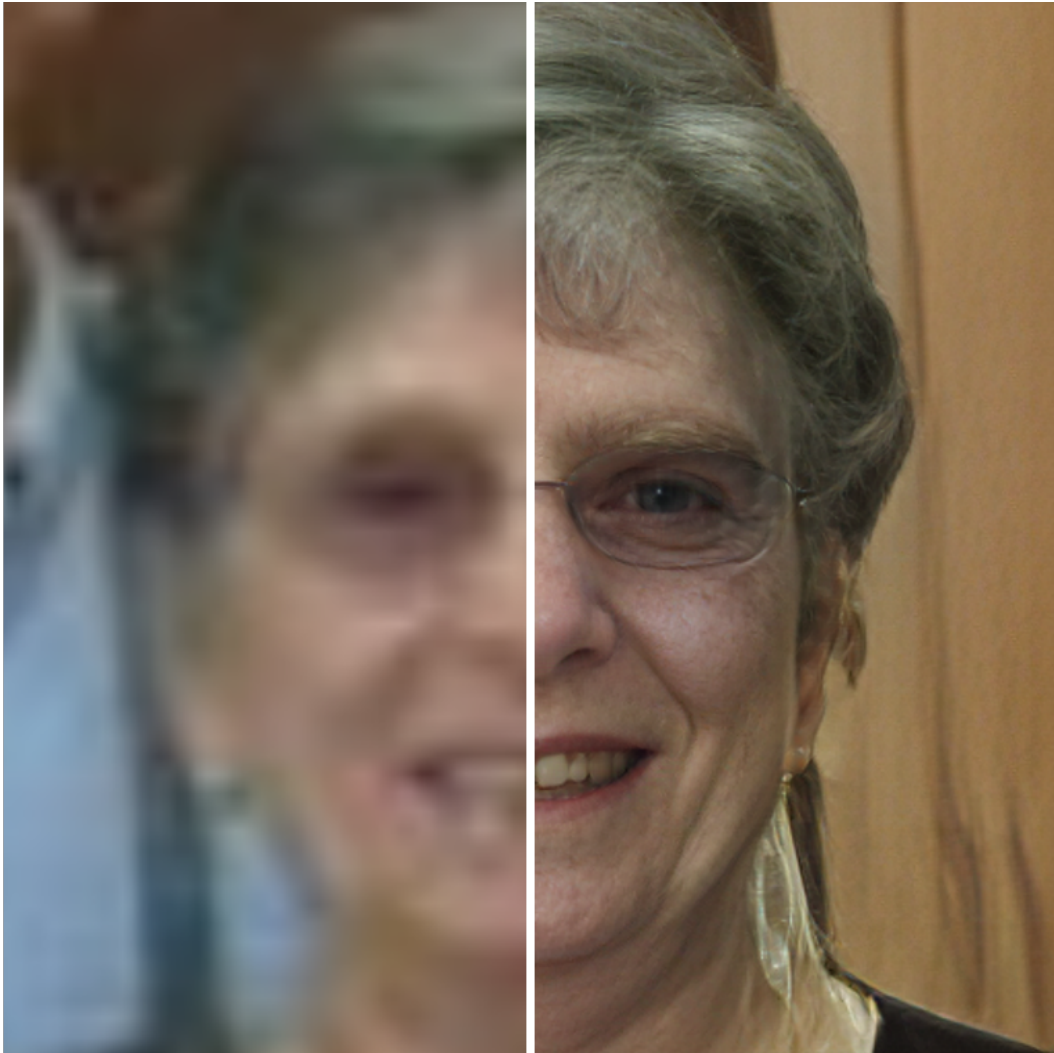

*Figure 19.* Blind face image restoration, with $512 \times 512$ resolution.

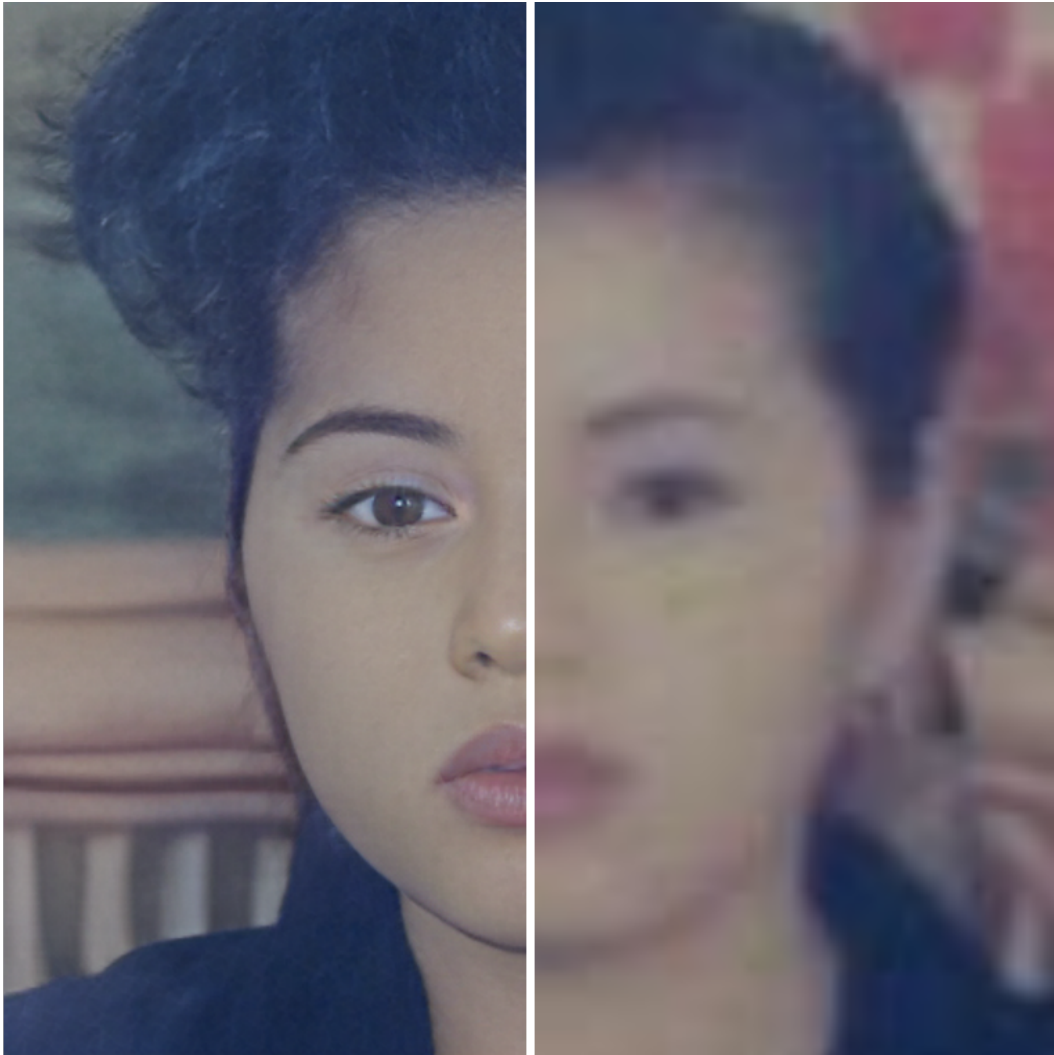

*Figure 20.* Blind face image restoration, with $512 \times 512$ resolution.

